# Discovery of the Hidden World with Large Language Models

**Chenxi Liu**[1]*, **Yongqiang Chen**[2,3,4]*, **Tongliang Liu**[5,2]
[1]TMLR Group, Hong Kong Baptist University
[2]Mohamed bin Zayed University of Artificial Intelligence
{cscxliu,bhanml}@comp.hkbu.edu.hk {yqchen,jcheng}@cse.cuhk.edu.hk

**Mingming Gong**[6,2]**, James Cheng**[4]**, Bo Han**[1]†**, Kun Zhang**[2,3]
[3]Carnegie Mellon University [4]The Chinese University of Hong Kong
[5]Sydney AI Centre, The University of Sydney [6]The University of Melbourne
tongliang.liu@sydney.edu.au mingming.gong@unimelb.edu.au kunz1@cmu.edu

**https://causalcoat.github.io**

## Abstract

Revealing the underlying causal mechanisms in the real world is the key to the development of science. Despite the progress in the past decades, traditional causal discovery approaches (CDs) mainly rely on high-quality measured variables, usually given by human experts, to find causal relations. The *lack of well-defined high-level variables in many real-world applications* has already been a longstanding roadblock to a broader application of CDs. To this end, this paper presents **C**ausal representati**O**n **A**ssistan**T** (COAT) that introduces large language models (LLMs) to bridge the gap. LLMs are trained on massive observations of the world and have demonstrated great capability in extracting key information from unstructured data. Therefore, it is natural to employ LLMs to assist with proposing useful high-level factors and crafting their measurements. Meanwhile, COAT also adopts CDs to find causal relations among the identified variables as well as to provide feedback to LLMs to iteratively refine the proposed factors. We show that LLMs and CDs are mutually beneficial and the constructed feedback provably also helps with the factor proposal. We construct and curate several synthetic and real-world benchmarks including analysis of human reviews and diagnosis of neuropathic and brain tumors, to comprehensively evaluate COAT. Extensive empirical results confirm the effectiveness and reliability of COAT with significant improvements.

## 1 Introduction

Science originates along with identifying important variables and revealing their causal relations [1, 2]. Despite the progress in the past decades, existing causal discovery approaches (CDs) mainly rely on high-quality measured variables, which are usually given by human experts [3, 4, 5]. However, the desired variables and their measurements are usually unavailable in a wide range of real-world applications. For example, Amazon sellers who want to analyze the factors related to user ratings only have raw user reviews, which are written according to the underlying user preferences for certain product characteristics. Therefore, *the lack of high-quality high-level variables* has been a longstanding impediment to broader real-world applications of CDs or causality-inspired methods [6].

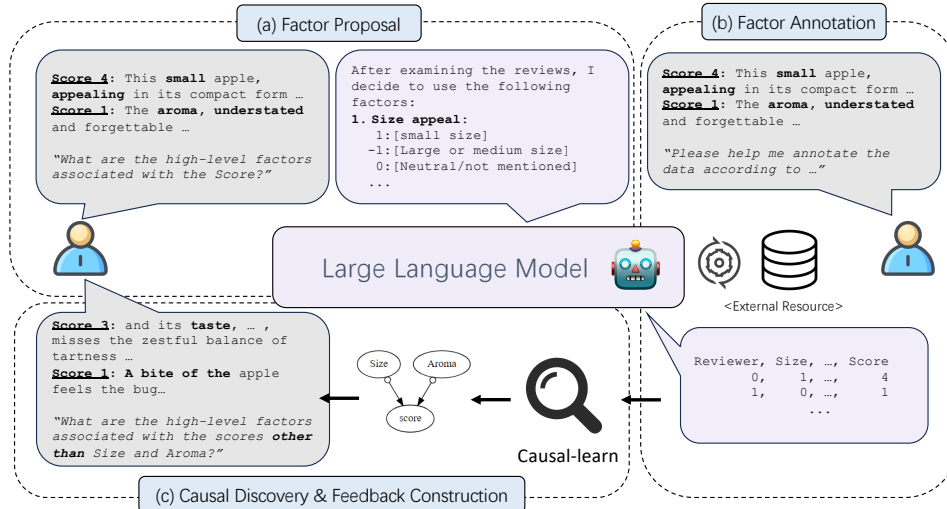

Figure 1: Illustration of COAT framework to analyze the rating scores of AppleGastronome. COAT aims to uncover the underlying Markov Blanket with respect to the given ratings of the apples (i.e., factors that fit the preferences of gastronomes). COAT first (a) adopts an LLM to read, comprehend, and relate the rich knowledge about tasting the apples. The LLM needs to propose a series of candidate factors such as apple sizes and smells, along with some meta-information such as annotation guidelines. Based on the candidate factors, COAT then (b) prompts another LLM to annotate the unstructured review into structured data. (c) The CD algorithm then finds causal relations among the factors, and constructs feedback based on samples where the ratings can not be well explained by the existing factors. By looking into the new samples, the LLM is expected to associate more related knowledge to uncover more desired causal factors.

The recently emerged Large Language Models (LLMs) [7, 8, 9, 10] offer a new opportunity to mitigate the gap [11]. Trained from massive observations of the world, LLMs demonstrate impressive capabilities in *comprehending unstructured inputs*, and leveraging the learned rich knowledge to resolve a variety of general tasks [12]. A surge of early tests demonstrates promising results that LLMs can effectively leverage the learned knowledge to answer commonsense causal questions [11, 13, 14]. Nevertheless, existing approaches mainly focus on incorporating LLMs as a *straightforward reasoner* with respect to the given *causal variables*. The reliability of LLMs in directly reasoning the true causal structure behind any specific data-generating process remains a debate [13, 15, 16, 17] due to a series of drawbacks of LLMs [18, 19, 20]. In addition, all of the existing combinations of LLMs and causal discovery have surprisingly overlooked the identifiability of causal structure, which plays an important role in classic causal discovery literature [3, 4, 5]. Hence, a challenging question comes:

*How can LLMs reliably assist in revealing the causal mechanisms behind the real world?*

In this work, we answer the question with a focus on local causal discovery with respect to a target variable that poses high value such as customer ratings and medical diagnosis, and introduce **C**ausal representati**O**n **A**ssistan**T** (COAT). Specifically, given the target variable $Y$, COAT aims to identify a Markov blanket to $Y$ from raw observation and also produce the theoretical-guaranteed causal structure. To achieve the goal, as illustrated in Fig. 1, COAT employs two mutually beneficial components: LLMs and CDs. Iteratively, at step (a), COAT leverages LLMs to look into a set of unstructured observations (e.g., customer comments) and propose potentially useful high-level factors. The proposed factors contain both the definition and the annotation criteria. Therefore, at step (b), another LLM is employed to give concrete values following the criteria. Then in step (c), CDs is used to reveal the structure among the identified factors. To ensure the reliability of the factor identification, COAT constructs feedback from the intermediate causal discovery results from step (c) to further inspire LLMs to improve further factors. The feedback includes sampling important observations that can not be well explained by the existing identified results. We show that the feedback provably helps with identifying the desired Markov Blanket and the structure (Proposition 2.2). We present a comprehensive analysis of COAT on both synthetic simulations and

real-world case studies, ranging from analysis of human reviews to diagnosis of neuropathic and brain tumors (Sec. 4). Our contributions are summarized as follows:

- To the best of our knowledge, we are the first to leverage LLMs to propose high-level variables, thereby extending the scope of CDs to unstructured data (Sec. 2.1).

- We establish the first benchmarks with real-world data including AppleGastronome and Neuropathic to examine the unstructured causal discovery (Sec. 4).

- We propose the first framework COAT that combines the best of LLMs and CDs to find theoretically grounded causal results (Sec. 2.2), which are verified with extensive empirical studies (Sec. 4).

- Additionally, the analysis of COAT also derives the first metrics that measure the causal representation learning capabilities of various LLMs (Definition 2.3).

## 2 Representation Assistant for Causal Discovery

In this section, we present the formulation of leveraging a language model to serve as a representation assistant for causal discovery on unstructured data. The representation assistant needs to extract useful factors that capture sufficient information for an interested target variable.

### 2.1 Problem Definition

**Data** To begin with, we are given a *target variable* of interest $Y$, e.g., stars rated by a customer or the tumor type of a patient. We treat $Y$ as a scalar random variable without loss of generality. The *unstructured data*, or raw observations, e.g., customer review of a certain product or images of tumors, are denoted as $\boldsymbol{X}$. The dataset $\mathcal{D}$ consists of $n$ paired samples $\{(\boldsymbol{x}_i, y_i)\}_{i=1}^{n}$ that are independently drawn from the distribution over $(\boldsymbol{X}, Y)$. Note that the target variable $Y$ serves as a guider, and no prior assumption on the relations between $\boldsymbol{X}$ and $Y$ is assumed.

**Objective** We seek a mapping $h : \boldsymbol{X} \mapsto \boldsymbol{Z}$ which elicits the *structured* representation $\boldsymbol{Z} = h(\boldsymbol{X})$ such that $Y \perp\!\!\!\perp \boldsymbol{X} \mid \boldsymbol{Z}$. In other words, $\boldsymbol{Z}$ serves as a Markov Blanket of $Y$ for the unstructured raw observations. Built upon the structure representation, then, downstream causal discovery methods can be applied on $\boldsymbol{Z} \cup \{Y\}$. The revealed causal structure can provide insights about the target variable $Y$ [21, 22], such as what factors of the product would be most satisfactory to customers. Furthermore, the framework can be easily extended to discover a complete causal graph by shifting the target variable to the other identified factors or the other additionally available variables. Formal assumptions are discussed in Sec. 2.4.

**LLM as a representation assistant** We aim to make the most use of the rich knowledge of LLMs to assist in extracting the relevant information from the raw observations $\boldsymbol{X}$. To this end, the mapping $h$ is decomposed as a collection of factors $\mathcal{W} = \{\boldsymbol{w}_1, \boldsymbol{w}_2, ..., \boldsymbol{w}_k\}$, each of which is a function $\boldsymbol{w}_i : \boldsymbol{X} \mapsto \mathcal{C}$ that maps the raw observation $\boldsymbol{x}$ to a predefined value space $\mathcal{C}$. In other words, the structured representation is composed of multiple factors: $h(\boldsymbol{X}) = (\boldsymbol{w}_1(\boldsymbol{X}), \boldsymbol{w}_2(\boldsymbol{X}), \cdots, \boldsymbol{w}_k(\boldsymbol{X}))$. Throughout this work, for the notation of factors, we use the symbol $\boldsymbol{w}_i$ to denote the factor itself like *sweetness*, *size*, or *scent*, and $\boldsymbol{w}_i(\cdot)$ to denote the function that maps from raw observation space $\mathcal{X}$ to the predefined value space $\mathcal{C}$.

**Descriptions of the factors** Without loss of generality, in this work, we consider the descriptions of the factors in natural language, which can be divided into two categories: i) Implicit factors, which need to be discovered and elaborated by LLMs. To obtain the values of the implicit factors, one could feed the factor descriptions and the unstructured input $\boldsymbol{x}_i$ to a suitable LLM for value annotation; For example, given a customer review on an apple $\boldsymbol{x}_i$, a discovered implicit factor $\mathcal{C} = \{-1, 0, 1\}$: $\boldsymbol{w}_1(\boldsymbol{x}_i) = 1$ could mean that the customer appreciates the sweetness of the apple; $\boldsymbol{w}_1(\boldsymbol{x}) = -1$ means that the customer is disappointed about the sweetness; $\boldsymbol{w}_1(\boldsymbol{x}) = 0$ means the sweetness has not been mentioned. ii) Explicit factors such as heart rate, whose descriptions are already available. The measure of the explicit factors usually requires some external tools.

### 2.2 The COAT Framework

We approach the aforementioned problem via a new framework called **C**ausal representati**O**n **A**ssistan**T** (COAT) (Algorithm 1). COAT aims to extract useful factors through multiple rounds of

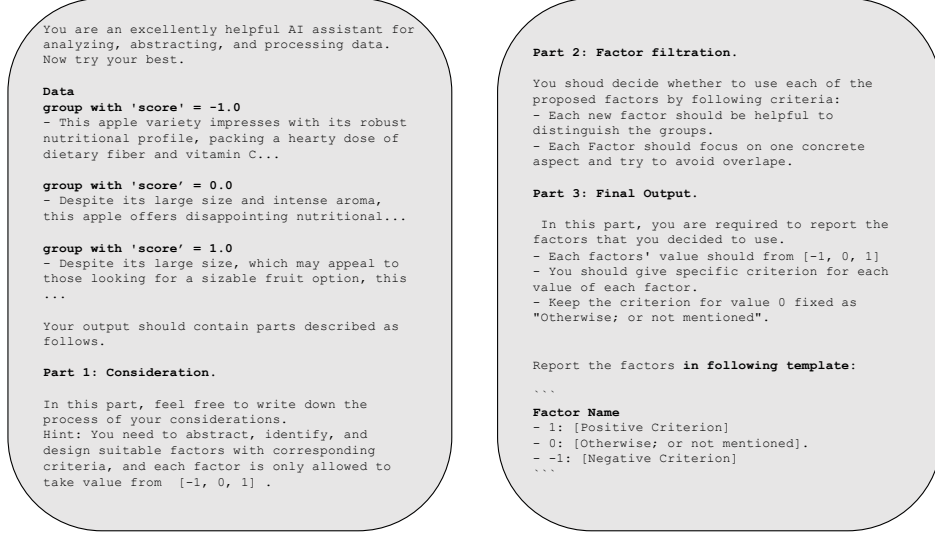

Figure 2: Illustration of the prompt template for factor proposal in COAT.

iteration. We use the superscript $t$ to refer to the input and output of LLM at the $t$-th round. We also denote the union of results from the first $t$ rounds using the superscript "$\leq t$".

**Factor proposal** To induce useful high-level factors from the rich world knowledge of LLMs, COAT employs a prompt $\boldsymbol{p}$ and a few samples $\widehat{\mathcal{D}} \subseteq \mathcal{D}$.

Specifically, in Fig. 2, $\boldsymbol{p}$ contains three components: *samples*, *instructions*, and *format control*. To encourage LLM to focus on the information related to the target variable $Y$, samples are grouped by the value of $Y$. The instruction requires $\Psi$ to give each proposed factor $\boldsymbol{w}_i$ a concrete description of the mapping $\boldsymbol{w}_i(\cdot)$, like how to decide the factor values. In addition, the metadata about the task such as the task description and context can also be incorporated if available. The prompt $\boldsymbol{p}$ essentially imitates human experts [23] in selecting and

---

**Algorithm 1** The COAT Framework

1: **Required:** Dataset $\mathcal{D} = \{(\boldsymbol{x}_i, y_i)\}_{i=1}^n$; LLM for factor proposal $\Psi$; Model for factor parsing $\Psi_s$; causal discovery algorithm $\mathcal{A}$; Feedback constructor $\mathcal{F}$; Maximal rounds $T$;
2: Random sampling $\widehat{\mathcal{D}}^1$;
3: Constructing $\boldsymbol{p}^1$;
4: **while** not converge and current round $t < T$ **do**
5:    $\mathcal{W}^t \leftarrow \Psi(\boldsymbol{p}^t, \widehat{\mathcal{D}}^t)$; //factor proposal
6:    $\boldsymbol{Z}^t \leftarrow \Psi_s(\mathcal{D}, \mathcal{W}^t, \boldsymbol{p}_p)$; //factor parsing
7:    $\mathcal{G}^t \leftarrow \mathcal{A}(\boldsymbol{Z}^{\leq t} \cup \{Y\})$; //causal discovery
8:    $(\widehat{\mathcal{D}}^{t+1}, \boldsymbol{p}^{t+1}) \leftarrow \mathcal{F}(\mathcal{G}^t, \mathcal{D}, \boldsymbol{p}^t)$; //feedback
9: **end while**
10: **return** $\mathcal{G}^T$

---

defining high-level variables. The set of resulting factors in the $t$-th round, defined with natural language by the LLM $\Psi$, is denoted as $\mathcal{W}^t = \Psi(\boldsymbol{p}^t, \widehat{\mathcal{D}}^t)$. We merge them with factors proposed in the previous rounds to update the set of all factors $\mathcal{W}^{\leq t} = \mathcal{W}^1 \cup \cdots \cup \mathcal{W}^t$.

**Factor parsing** Once we obtain the candidate factors, we then collect the values of the factors from the unstructured observations. In prior works, they are usually collected from human experts according to the given factors [3]. To do so, another LLM $\Psi_s$ is instructed to read the annotated guidelines of each variable $\boldsymbol{w}_i$ and parse the unstructured observations into structured or tabular data:

$$\boldsymbol{z}_i := \big(\boldsymbol{w}_i(\boldsymbol{x}_1), \cdots, \boldsymbol{w}_i(\boldsymbol{x}_n)\big), \quad \boldsymbol{w}_i(\boldsymbol{x}) := \Psi_s(\boldsymbol{x}, \boldsymbol{w}_i, \boldsymbol{p}_p), \qquad (1)$$

where $\boldsymbol{p}_p$ refers to the additional instruction to prompt $\Psi_s$ to parse the observed data, and $\boldsymbol{z}_i$ refers to the parsed values for the corresponding factor $\widehat{\boldsymbol{w}}_i$. We define $\boldsymbol{Z}^{\leq t} := \text{Concat}\big(\{\boldsymbol{z}_i \text{ for } \boldsymbol{w}_i \in \mathcal{W}^{\leq t}\}\big)$.

When the data curation of the proposed factors requires additional domain-specific knowledge/skills (e.g., intervening on the external environments) that the LLMs do not acquire, we could fetch $\boldsymbol{z}_i$ through some external process [24, 25]. For example, studying the causes of a disease requires annotating relevant symptoms from diagnosis records and conducting additional medical checks [26]. Our experiments show that COAT can effectively extract the hidden factors under both schemes.

**Causal discovery** With the given values $\boldsymbol{Z}^{\leq t}$ associated with the candidate factors $\mathcal{W}^{\leq t}$, a CD algorithm $\mathcal{A}$ (e.g., FCI [27]) is used to reason about the causal structure based on the parsed data:

$$\mathcal{G}^t = \mathcal{A}(\boldsymbol{Z}^{\leq t} \cup \{Y\}), \tag{2}$$

where $\mathcal{G}^t$ is the discovered causal structure. In general, the inputs in each round to $\mathcal{A}$ may contain noises as well as latent confounders, any CDs with suitable theoretical assumptions could be used for $\mathcal{A}$. The noises injected through LLM-based parsing may be of independent interest to the literature of causal discovery [5, 28]. In this work, we demonstrate the idea of COAT via the FCI algorithm [4] as it is flexible with respect to different functional classes of the underlying generation process, allows for the existence of latent confounders [27], which aligns well with our objective.

**Improving factor proposal with causal feedback** LLMs require proper prompts to fully unlock their capabilities [29, 30, 31]. When it comes to factor proposing, it is also hard for LLMs to propose all factors at once. Nevertheless, from the causal discovery results, we could find useful information and thus provide feedback to further improve the factor proposal:

$$(\widehat{\mathcal{D}}^{t+1}, \boldsymbol{p}^{t+1}) = \mathcal{F}(\mathcal{G}^t, \mathcal{D}, \boldsymbol{p}^t), \tag{3}$$

where $\mathcal{F}$ samples specific examples from $\mathcal{D}$ and constructs new prompts according to the results of $\mathcal{A}$ for the next round of factor proposal. For example, FCI is able to imply the existence of latent confounders, from which we could refine $\boldsymbol{p}$ to prompt $\Psi$ to focus on the corresponding factors.

## 2.3 Causal Feedback

Let $X$ be the high-dimensional random variable for the raw data. After $t$ rounds, COAT identifies $h_{\leq t}(X)$ as the Markov Blanket for $Y$ w.r.t. to $\{\boldsymbol{w}_i(X) \mid \boldsymbol{w}_i \in \mathcal{W}^{\leq t}\}$. If $Y \not\perp\!\!\!\perp X \mid h_{\leq t}(X)$, which means it cannot serve as a Markov Blanket [32] for $Y$ w.r.t. $X$, then there exists a potential factor $\widehat{\boldsymbol{w}} : X \mapsto \mathcal{C}$ such that:

$$H(Y|h_{\leq t}(X)) > H(Y|h_{\leq t}(X), \widehat{\boldsymbol{w}}(X)), \tag{4}$$

where $H(\cdot)$ refers to the entropy. If the LLM $\Psi$ can not find the desired $\widehat{\boldsymbol{w}}$, it means that $h_{\leq t}(X)$ is already sufficient to separate $Y$ from $X$. Therefore, for the next $(t+1)$-th iteration with sample $\widehat{\mathcal{D}}^{t+1}$, $\Psi$ is expected to propose new factor $\widehat{\boldsymbol{w}}$ that also satisfies the similar property:

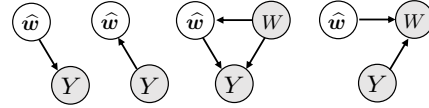

Figure 3: Illustration of variables that could be discovered with COAT. Let $W \in h_{\leq t}(X)$ be an identified variable, and assume there exists a latent variable $\widehat{w}$ to be discovered. When $\widehat{w}$ is the direct parent or child of $Y$, finding hard-to-explain samples can help uncover it. When $\widehat{w}$ is the direct parent and also a child of $W$, or the spouse of $Y$ with $W$ as the common child of $Y$ and $\widehat{w}$, conditioning on $W$ facilitates the discovery of $\widehat{w}$.

$$H_{\widehat{\mathcal{D}}^{t+1}}(Y|h_{\leq t}(X)) - H_{\widehat{\mathcal{D}}^{t+1}}(Y|h_{\leq t}(X), \widehat{\boldsymbol{w}}(X)) > 0, \tag{5}$$

where $H_{\widehat{\mathcal{D}}^t}(\cdot|\cdot)$ refers to the conditional entropy measured on $\widehat{\mathcal{D}}^t$. As shown in Fig. 3, finding factors satisfying Eq. 4 progressively expands the discovered factors and pushes $h_{\leq t}(X)$ to a valid Markov Blanket. Also, it implies conditioning on the identified factors would further strengthen the correlation between $\widehat{w}$ and $Y$. Therefore, to find the desired factor, we are motivated to select suitable $\widehat{\mathcal{D}}^{t+1}$ for the next iteration such that $\widehat{\mathcal{D}}^{t+1} = \arg\max_{\widehat{\mathcal{D}} \subseteq \mathcal{D}} H_{\widehat{\mathcal{D}}}(Y|h_{\leq t}(X))$, where $h_{\leq t}(X)$ can not well explain $Y$. This problem can be converted into a classification problem in which $\widehat{\mathcal{D}}^*$ are the samples that the fitted classifier yields a large prediction error. In our experiments, we implement the classification via clustering with respect to $h_{\leq t}(X)$. The clustering elicits $C$ groups $\widehat{\mathcal{D}}_c := \{\boldsymbol{x}_i \text{ for } i \in \mathcal{I}_c\}$: $\mathcal{I}_1, \cdots, \mathcal{I}_C = \text{K-Means}(h_{\leq t}(X))$. We then take the group of samples with the largest conditional entropy to construct the feedback.

In practice, many factors, such as the LLM capabilities, data faithfulness, and prompt templates, could affect the satisfaction of Eq 5. Therefore, in the next section, we will establish a theoretical framework to discuss the influence of the factors above to the satisfaction of Eq 5.

## 2.4 Theoretical Analysis

We then theoretically analyze some critical steps in COAT, including the feedback and identifiability.

**Feedback analysis** Given a new factor $\boldsymbol{w}_{k+1}$, with the current representation as $h_{[k]}(X) = (\boldsymbol{w}_1(X), \boldsymbol{w}_2(X), \cdots, \boldsymbol{w}_k(X))$, and COAT tests:

$$Y \not\perp\!\!\!\perp \boldsymbol{w}_{k+1}(X) \mid h_{[k]}(X). \tag{6}$$

COAT also requires the following usual condition about distribution and causal graph:

**Assumption 2.1** (Faithful and Markov conditions, adjusted from [33]). *For any disjoint non-empty subsets $A, B, C \subset \mathcal{W}^{\leq t} \cup \{Y\}$, $A$ and $B$ are d-separated by $C$ on the causal graph iif $A \perp\!\!\!\perp B \mid C$ on the factors' distribution. All conditional independencies are preserved after factor parsing.*

The annotation from a poor model could introduce an additional "error term" on the true factor values, disturbing the true distribution, as one can observe in Fig. 4(a) and 4(b). If Assumption 2.1 holds, the conditional mutual information between $Y$ and $X$ given the desired factors decreases:

**Proposition 2.2.** *Under assumption 2.1, if condition 6 holds, then for Markov Blanket $\mathcal{S} \subseteq [k+1]$ of $Y$, i.e., $Y \perp\!\!\!\perp h_{[k+1]\setminus\mathcal{S}}(X) \mid h_{\mathcal{S}}(X)$, we have the following about conditional mutual information:*

$$I\left(Y; X \mid h_{\mathcal{S}}(X)\right) = I\left(Y; X \mid h_{[k+1]}(X)\right) < I\left(Y; X \mid h_{[k]}(X)\right) \tag{7}$$

**Factor identification** We provide an initial exploration under what conditions LLMs can identify target-related factors and how the ability of an LLM influences this procedure.

**Definition 2.3** (Ability of LLMs). *Given a suitable prompt about current factors and data, the LLM $\Psi$ has non-zero probability $p_\Psi > 0$ to propose a new factor $\boldsymbol{w}_{k+1}$ that satisfies condition 6 and*

$$\frac{I\left(Y; X \mid h_{[k+1]}(X)\right)}{I\left(Y; X \mid h_{[k]}(X)\right)} < 1 - C_\Psi, \tag{8}$$

*for some positive constant $C_\Psi$ whenever $I\left(Y; X \mid h_{[k]}(X)\right) > 0$. Note that $h_{[0]}(X) = \phi$, hence we also use $I\left(Y; X \mid h_{[0]}(X)\right)$ to refer $I\left(Y; X\right)$. We use $p$ instead of $p_\Psi$ when the context is clear.*

We further explain the intuition behind Def 2.3: the *Perception Score $p$* captures the LLM's responsiveness to the given prompts and the feedback; the *Capacity Score $C_\Psi$* captures the quality of the factors proposed by the LLM. Empirically, the two scores are used to estimate the abilities of the predominant LLMs (Sec. 3.2). Theoretically, we use them to characterize the influence of prompt templates, the LLM responsiveness, and the quality of factors on the performance of COAT:

**Proposition 2.4** (Characterization for Factor Identification Process). *With assumption 2.1, for any small number $\epsilon, \delta \in (0, \frac{1}{2})$, perception score $p > 0$, capacity score $C_\Psi > 0$, with $t$ COAT rounds that*

$$\sqrt{t} > \frac{|z_\delta|\sqrt{1-p}}{2\sqrt{p}}\left(1 + \sqrt{1 + \frac{4\log\epsilon}{z_\delta^2(1-p)\log(1-C_\Psi)}}\right), \tag{9}$$

*where $z_\delta$ is the $\delta$-quantile of the standard normal distribution, we have*

$$\Pr\left(\frac{I\left(Y; X \mid h_{\leq t}(X)\right)}{I\left(Y; X\right)} < \epsilon\right) \geq 1 - \delta. \tag{10}$$

The proof is given in Appendix D.2. Prop. 2.4 gives a guarantee on identifying a Markov Blanket. Intuitively, Prop. 2.4 also characterizes the influence of prompt templates, the LLM responsiveness, and the quality of factors on the performance of COAT via the two proposed measures: $p$ and $C_\Psi$. When both of them are positive, COAT would converge exponentially:

**Proposition 2.5** (Rate of Convergence). *With assumption 2.1, for any small number $\epsilon, \delta \in (0, \frac{1}{2})$, perception score $p > 0$, capacity score $C_\Psi > 0$, after $t$ COAT rounds, the following inequality holds with probability at least $1 - \delta$:*

$$\frac{I(Y; X \mid h_{\leq t}(X))}{I(Y; X)} \leq \left(\frac{1}{1-C_\Psi}\right)^{-tp-z_\delta\sqrt{tp(1-p)}} \tag{11}$$

**Causal structure identification** It is clear that LLMs are not involved in the causal discovery process, which is mainly executed by causal discovery methods such as FCI. Therefore, the CD guarantees the identifiability of the final causal graph over the LLM-proposed factors. The concrete

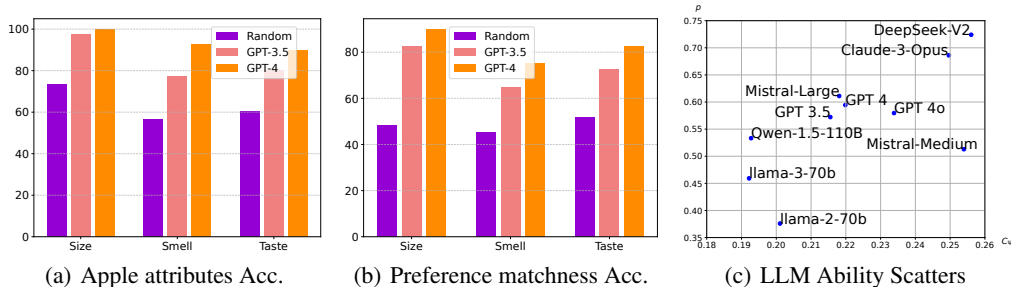

(a) Apple attributes Acc.  (b) Preference matchness Acc.  (c) LLM Ability Scatters

Figure 4: Quantitative evaluation of the causal capabilities of LLMs in COAT.

assumptions required for identifiability depend on the specific CD used in COAT. For instance, the FCI algorithm requires faithfulness of the data distribution with respect to the true causal graph [27]. In our experiments, we also verify that the structured data annotated by LLMs has a high accuracy and little noise, which is friendly to the CD assumption. In general, one could switch to another CD in COAT, while using different CDs may require different assumptions. For example, the LiNGAM algorithm requires the relations among variables to be linear and non-Gaussian models. Empirically, we find that COAT with LiNGAM works very well (Appendix F).

## 2.5 Practical Discussions

**Prompt template**  The instruction following capability and the context window of LLMs may affect the satisfaction to the constraints of the prompt template. Including more data samples or background knowledge may improve the $p$ and $C_\Psi$, but it is more challenging for the LLM.

**Modern causal discovery**  We use FCI in this paper in order to illustrate the idea. To attain identifiability better than the Markov equivalent class, one can choose more advanced Causal Discovery methods under different assumptions, see Appendix C if interested.

We also discuss some cases where we need to handle them properly in practice with LLMs.

**Factor filtering**  LLMs may output several factors with similar semantics or exhibit multicollinearity in the annotated data, which will hinder the causal discovery process. To mitigate the issue, one could do factor filtering that adopts PCA or early conditional independence tests given the currently discovered variables in the Markov Blanket to detect and eliminate these variables.

**Factor pool**  LLMs may discover useful factors in early rounds while being discarded. For example, the underlying spouse variables of the target label $y$ may be independent with $y$ without conditioning on their common children variables. To resolve the issue, we could introduce a factor pool that stores the candidate variables proposed in the past, and replay the variables that have not been passed by conditional independence tests with existing variables in the Markov Blanket for a double check.

## 3 Empirical Analysis of COAT

We evaluate whether COAT can propose and identify a set of high-level factors belonging to the Markov Blanket of the target variable $Y$. We construct the first benchmark, called AppleGastronome, to verify the effectiveness of COAT in finding useful causal information, and compare COAT to previous methods [11] that merely leverage LLMs to perform causal discovery. Specifically, we use AppleGastronome to examine the capabilities of 10 predominant LLMs including GPT 4o [34], Claude-3-Opus [35], LLaMA3-70b [36], and Mistral-Large [37] in realizing COAT. Due to the space limit, we report only the results of the popular LLMs and present full results in Appendix E.4.

### 3.1 Experimental Setup

In AppleGastronome benchmark, we consider the target variable as a rating score of the apple by astronomers. We prepare different high-level factors: 3 parents of Y, one child of Y, and one spouse of Y. These factors form a Markov blanket of Y. In addition, we also prepared one disturbing factor

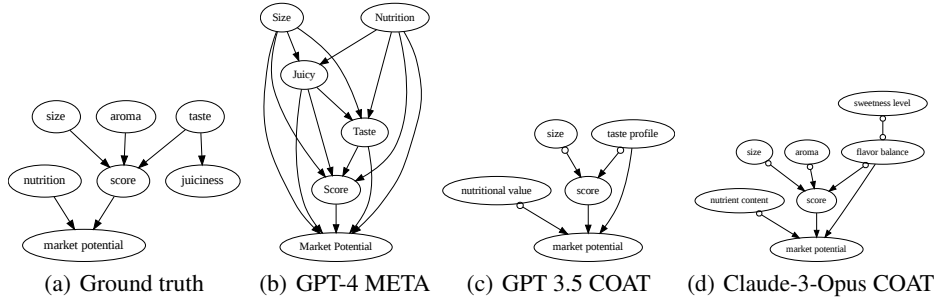

| (a) Ground truth | (b) GPT-4 META | (c) GPT 3.5 COAT | (d) Claude-3-Opus COAT |

Figure 5: The discovered causal graphs in AppleGastronome. Compared to the ground truth results, directly adopting LLMs to reason the causal relations can easily elicit many false positive edges. In contrast, the relations recovered by COAT have a high precision and recall. The directed edge between "taste" and "juiciness" can not be recovered by COAT because of the limitations of FCI.

related to Y but not a part of this blanket. A good method is expected to propose the five high-level factors (up to semantic meanings) and exclude the disturbing factor.

**Benchmark construction** Each apple has its own attributes, including size, smell, and taste (or sweetness). Each gastronomy pays unique attention to a subset of the above three attributes. They will write a review according to their preference and give the rating score. We generate the review using GPT 4 by feeding GPT 4 the preferences and the apple attributes. We generated 200 samples for LLMs' analysis and annotation. More details are left in Appendix E.1.

**Baselines** For factor proposal, we mainly employ two different uses of LLMs as the baselines: **META** is the zero-shot factor proposal given only the context to LLMs; and **DATA** additionally gives some samples of raw observations, which is an ablation of COAT without the *feedback* module, i.e., only one COAT round. For causal relation inference, we follow Kiciman et al. [11] that prompt LLMs to reason for the causal direction of each pair of the discovered variables by **DATA**.

**Metrics** We evaluate the ability on factor proposal based on three metrics: *MB*, *NMB*, and *OT*. *MB* means the desired factor forming the Markov Blanket of Y. *NMB* means the undesired factor relevant to data but not in *MB*. *OT* means the unexpected factors irrelevant to data. We also present the corresponding recall, precision, and F1 with respect to MB($Y$).

## 3.2 Analysis with AppleGastronome Benchmark

**Key findings** Empirically, LLMs with CoT can be aware of high-level factors behind data (lower *OT* than *META*) but still struggle to distinguish the desired factors in Markov Blanket (higher *NMB* than COAT). COAT is more resistant to the "disturbing" factor, which is supported by the lower *NMB* column. COAT filters out irrelevant factors from LLMs' prior knowledge that are not reflected by the data, which is

Table 1: Factor proposal results in Apple Gastronome benchmark (Full results in Appendix E.4).

| LLM | METHOD | FACTOR PROPOSAL | | | | |
| --- | --- | --- | --- | --- | --- | --- |
| | | MB | NMB | OT | RECALL | PRECISION | F1 |
| GPT 4 | META | 2.67±0.94 | 0.67±0.47 | 2.33±0.47 | 0.53±0.19 | 0.46±0.08 | 0.49±0.13 |
| | DATA | 3.00±0.00 | 0.33±0.47 | 0.00±0.00 | 0.60±0.00 | 0.92±0.12 | 0.72±0.04 |
| | DATA+CoT | 4.33±0.58 | 0.83±0.29 | 0.17±0.29 | 0.87±0.12 | 0.81±0.02 | 0.84±0.06 |
| | COAT | 4.00±0.82 | 0.33±0.47 | 0.00±0.00 | 0.80±0.16 | 0.93±0.09 | 0.85±0.11 |
| GPT 3.5 | META | 3.33±1.25 | 0.33±0.47 | 4.33±1.25 | 0.67±0.25 | 0.42±0.12 | 0.51±0.15 |
| | DATA | 2.67±0.47 | 0.67±0.47 | 0.00±0.00 | 0.53±0.09 | 0.81±0.14 | 0.64±0.10 |
| | DATA+CoT | 5.00±0.00 | 1.00±0.00 | 1.33±0.58 | 1.00±0.00 | 0.68±0.05 | 0.81±0.04 |
| | COAT | 3.67±0.47 | 0.00±0.00 | 0.00±0.00 | 0.73±0.09 | 1.00±0.00 | 0.84±0.07 |
| LLAMA2 -70B | META | 2.33±0.47 | 0.67±0.47 | 4.67±1.25 | 0.47±0.09 | 0.32±0.07 | 0.37±0.06 |
| | DATA | 2.33±0.94 | 0.67±0.47 | 0.00±0.00 | 0.47±0.19 | 0.75±0.20 | 0.57±0.20 |
| | DATA+CoT | 3.00±1.73 | 0.67±0.58 | 0.33±0.58 | 0.60±0.35 | 0.71±0.34 | 0.65±0.35 |
| | COAT | 3.00±0.00 | 0.67±0.47 | 0.00±0.00 | 0.60±0.00 | 0.83±0.12 | 0.69±0.04 |
| MISTRAL -MEDIUM | META | 3.00±0.00 | 0.67±0.47 | 1.67±1.25 | 0.60±0.00 | 0.59±0.13 | 0.59±0.07 |
| | DATA | 3.00±0.00 | 0.67±0.47 | 0.00±0.00 | 0.60±0.00 | 0.83±0.12 | 0.69±0.04 |
| | DATA+CoT | 4.33±0.58 | 1.00±0.00 | 0.67±0.58 | 0.87±0.12 | 0.73±0.07 | 0.79±0.05 |
| | COAT | 4.67±0.47 | 0.00±0.00 | 0.00±0.00 | 0.93±0.09 | 1.00±0.00 | 0.96±0.05 |

supported by the lower *OT* column. COAT robustly encourages LLM to find more expected factors through the feedback, which is supported by the higher *MB* column.

**Can LLMs be an effective factor proposer?** As discussed in Sec. 2.4, there are two crucial abilities for LLMs in identifying potential high-level factors. The first one is to be aware of the existence of potential factors, and the second is to synthesize and describe these factors. Inspired

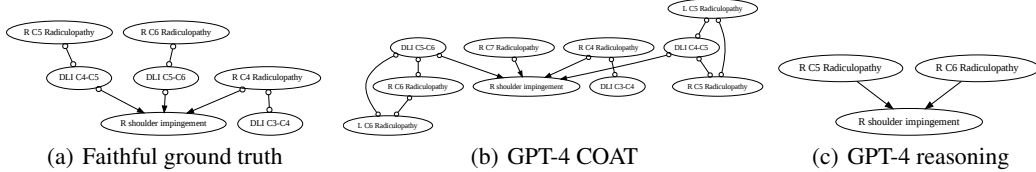

|                        |                       |                      |
|:----------------------:|:---------------------:|:--------------------:|
| (a) Faithful ground truth | (b) GPT-4 COAT | (c) GPT-4 reasoning |

Figure 6: The discovered causal graphs in Neuropathic. (c) shows the result based on directly prompting LLM to reason for the causal relations among all factors. Disconnected ones are dropped.

by this observation, we propose two novel metrics to quantify LLMs' causal ability: a) Perception that quantifies the ratio of *valid* factors (satisfying Prop. 2.2) proposed by LLMs in each round; b) Capacity that measures the effective mutual information drop in Assumption 2.3. As shown in Fig. 4(c), LLMs differ largely on the perception score while comparably on the capacity score.

**Can LLMs be an effective factor annotator?** As shown in Fig. 4, both GPT 3.5 and GPT 4 annotate subjective attributes well. Regarding objective human preferences, the performances are still relatively high. Empirically, LLMs will not introduce new confounders, see Appendix E.3.

**Can COAT reliably recover the causal relationships?** We present quantitative and qualitative results in Table 3 (in Appendix E.1) and Fig. 5 (on page 8), respectively. Compared to directly adopting LLMs to reason the causal relations, COAT significantly boosts the causal relation recovery. Meanwhile, COAT maintains high performances based on various LLMs, which further demonstrates the effectiveness of the causal feedback in COAT to improve the robustness of this system. In fact, the causal feedback focuses on making maximal use of the rich knowledge of LLMs, and reducing the reliance on the reasoning capabilities of different LLMs, to assist with causal discovery. We provide the full results of 10 LLMs in Appendix E.3.

## 4 Empirical Study with Realistic Benchmarks

After examining the capabilities of COAT in AppleGastronome, we are further motivated to challenge COAT in a more complex setting from neuropathic panic diagnosis [26], brain tumor detection with MRI images [38], three-years news summary about one stock from the New York Times [39], and climatic reanalysis data with fine-grained time and space coverage [40]. We refer to Appendix H for a complete summary of all five benchmarks.

### 4.1 Experimental Setup

**Benchmark construction** In the Neuropathic benchmark, we convert the dataset into a clinical diagnosis task. In the original dataset, there are three levels of causal variables, including the symptom level, radiculopathy level, and the pathophysiology level. In the experiments, we mainly consider the target variable of right shoulder impingement. When generating the clinical diagnosis notes as $x$ using GPT 4, we avoid any mentioning of variables other than symptoms. We generated 100 samples for LLMs' analysis; since the number of possible factors is finite, we generate 1000 tabular data for CI tests.

Table 2: Factor proposal results in Neuropathic. PA, AN, and OT refer to the parents, ancestors, and others, respectively. Accuracy and F1 measure the recovery of the causal ancestors.

| LLM | METHOD | FACTOR PROPOSAL | | | | |
|-----|--------|:--:|:--:|:--:|:----:|:----:|
|     |        | PA | AN | OT | ACC | F1 |
| GPT 4 | META | 3 | 5 | 6 | 0.91 | 0.59 |
|       | DATA | 2 | 2 | 0 | 0.95 | 0.50 |
|       | DATA+CoT | 3 | 4 | 13 | 0.81 | 0.35 |
|       | COAT | 3 | 6 | 3 | 0.96 | 0.80 |
| GPT 3.5 | META | 3 | 5 | 6 | 0.91 | 0.59 |
|         | DATA | 3 | 5 | 4 | 0.94 | 0.67 |
|         | DATA+CoT | 2 | 2 | 3 | 0.91 | 0.36 |
|         | COAT | 3 | 5 | 2 | 0.96 | 0.77 |
| LLAMA2 -70B | META | 2 | 4 | 5 | 0.91 | 0.53 |
|             | DATA | 3 | 3 | 1 | 0.95 | 0.60 |
|             | DATA+CoT | 2 | 4 | 7 | 0.88 | 0.47 |
|             | COAT | 3 | 6 | 2 | 0.97 | 0.86 |
| LLAMA2 -13B | META | 1 | 3 | 6 | 0.88 | 0.40 |
|             | DATA | 3 | 6 | 4 | 0.95 | 0.75 |
|             | DATA+CoT | 0 | 1 | 10 | 0.81 | 0.12 |
|             | COAT | 3 | 6 | 2 | 0.97 | 0.86 |
| LLAMA2 -7B | META | 1 | 1 | 17 | 0.72 | 0.08 |
|            | DATA | 3 | 6 | 3 | 0.96 | 0.80 |
|            | DATA+CoT | 0 | 0 | 10 | 0.79 | — |
|            | COAT | 3 | 6 | 2 | 0.97 | 0.86 |
| MISTRAL -MEDIUM | META | 3 | 6 | 3 | 0.96 | 0.80 |
|                 | DATA | 3 | 3 | 2 | 0.94 | 0.66 |
|                 | DATA+CoT | 3 | 5 | 8 | 0.88 | 0.53 |
|                 | COAT | 3 | 6 | 2 | 0.97 | 0.86 |

As we intend to leverage the Neuropathic benchmark to simulate the real-world diagnosis, after the factor proposal stage, we directly incorporate external tools to measure the values of the candidate factors. More details about the construction of the Neuropathic are given in Appendix E.6.

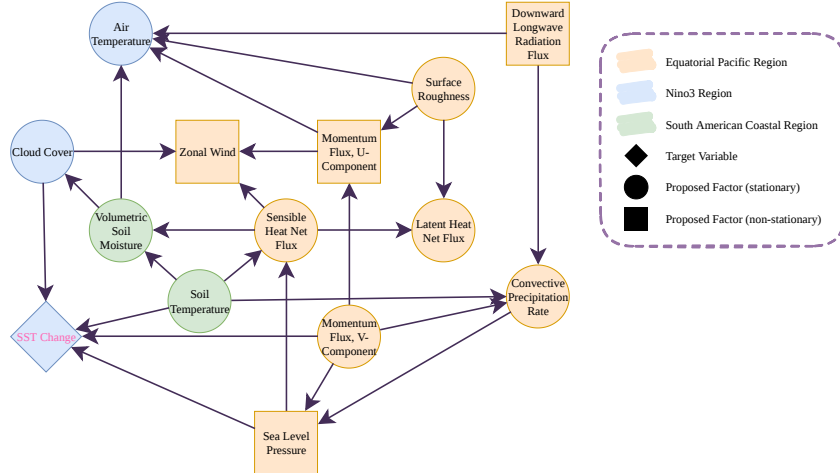

Figure 7: The final causal graph found by COAT in the ENSO case study

**Evaluation and baselines**    In Neuropathic, we adopt a similar evaluation protocol and the baselines as in AppleGastronome. Nevertheless, due to the faithfulness issue of the original dataset [26], for the evaluation of causal relation discovery, we mainly conduct a qualitative comparison between the ground truth that is faithful to the data, against the baselines and COAT.

## 4.2    Empirical Results on Neuropathic Benchmark

**Factor proposal**    The quantitative results on Neuropathic benchmark are given in Table 2. Similarly, we can find that COAT consistently outperforms all of the baselines regardless of which LLMs are incorporated. In particular, even with the weakest backbone model, i.e.,LLaMA2-7b, COAT can still effectively leverage the intrinsic rich knowledge and beat the baselines with more powerful LLMs.

**Causal relation recovery**    Fig. 6(a) shows the causal graph obtained by FCI running on the original data, where we can find that several causal relations cannot hold on the data. As shown in Fig. 6, when using LLMs to perform the reasoning, LLMs cannot identify the faithfulness issues. In contrast, COAT can imply faithful causal insights.

## 4.3    More Real-world Results

**El Niño–Southern Oscillation (ENSO) case study**    El Niño-Southern Oscillation (ENSO) is a climatic phenomenon in the Pacific Ocean that influences global weather patterns profoundly. To understand its mechanism, we apply COAT on NOAA dataset [40]. There are 13 factors identified by COAT, and their instantaneous causal relations are visualized in Fig 7. The target variable is the future change in monthly SST in the Nino3 region, which could be an important indicator of ENSO events. Each factor is a time series about a certain climate measurement above a specific level averaged over a specific region. The paths about *Sea level Pressure*, *Momentum Flux*, and *Cloud Cover* matches the existing understanding from literature [41, 42, 43, 44]. It also suggests several inserting hypotheses that are less explored in literature, like the path from Soil Temperature in South American Coastal Region. We refer details in Appendix K.

**More real-world empirical studies**    We also report concrete results on real-world problems involving MRI, time series, and NetCDF data in Appendix I, J, and K.

## 5    Conclusions

In this paper, we proposed a new paradigm COAT to incorporate the rich knowledge of LLMs into the CD pipeline. We found that COAT effectively extends the scope of CDs to unstructured data by identifying useful high-level variables from raw observations for CD methods. COAT suggests a new pathway towards building a causal foundation model for discovery. We leave more detailed discussions about future studies in Appendix B.

## Acknowledgments and Disclosure of Funding

We thank the reviewers for their valuable comments. This material is based upon work supported by NSF Award No. 2229881, AI Institute for Societal Decision Making (AI-SDM), the National Institutes of Health (NIH) under Contract R01HL159805, and grants from Salesforce, Apple Inc., Quris AI, and Florin Court Capital. CXL and BH were supported by NSFC General Program No. 62376235, Guangdong Basic and Applied Basic Research Foundation Nos. 2022A1515011652 and 2024A1515012399, HKBU Faculty Niche Research Areas No. RC-FNRA-IG/22-23/SCI/04, and HKBU CSD Departmental Incentive Scheme. MMG was partially supported by the following Australian Research Council projects: DE210101624 and DP240102088. JC was supported by CUHK direct grant 4055146.

## Footnotes

*These authors contributed equally. †Correspondence to Bo Han (bhanml@comp.hkbu.edu.hk).

## References

[1] Norwood Russell Hanson. *Patterns of discovery : an inquiry into the conceptual foundations of science*. Cambridge University Press, 1958. (Cited on page 1)

[2] Thomas S. Kuhn and David Hawkins. The structure of scientific revolutions. *American Journal of Physics*, 31:554–555, 1963. (Cited on page 1)

[3] Peter Spirtes, Clark Glymour, and Richard Scheines. *Causation, Prediction, and Search, Second Edition*. Adaptive computation and machine learning. MIT Press, 2000. ISBN 978-0-262-19440-2. (Cited on pages 1, 2, 4, 21 and 22)

[4] Peter Spirtes, Clark Glymour, Richard Scheines, and Robert Tillman. *Automated Search for Causal Relations: Theory and Practice*, 2018. (Cited on pages 1, 2, 5, 21 and 22)

[5] Matthew J. Vowels, Necati Cihan Camgoz, and Richard Bowden. D'ya like dags? a survey on structure learning and causal discovery. *ACM Computing Survey*, 55(4), 2022. ISSN 0360-0300. (Cited on pages 1, 2, 5, 21 and 22)

[6] Bernhard Schölkopf, Francesco Locatello, Stefan Bauer, Nan Rosemary Ke, Nal Kalchbrenner, Anirudh Goyal, and Yoshua Bengio. Towards causal representation learning. *arXiv preprint*, arXiv:2102.11107, 2021. (Cited on pages 1, 21 and 22)

[7] Tom B. Brown, Benjamin Mann, Nick Ryder, Melanie Subbiah, Jared Kaplan, Prafulla Dhariwal, Arvind Neelakantan, Pranav Shyam, Girish Sastry, Amanda Askell, Sandhini Agarwal, Ariel Herbert-Voss, Gretchen Krueger, Tom Henighan, Rewon Child, Aditya Ramesh, Daniel M. Ziegler, Jeffrey Wu, Clemens Winter, Christopher Hesse, Mark Chen, Eric Sigler, Mateusz Litwin, Scott Gray, Benjamin Chess, Jack Clark, Christopher Berner, Sam McCandlish, Alec Radford, Ilya Sutskever, and Dario Amodei. Language models are few-shot learners. In *Advances in Neural Information Processing Systems*, 2020. (Cited on pages 2, 21 and 22)

[8] OpenAI. Chatgpt. https://chat.openai.com/chat/, 2022. (Cited on pages 2, 21 and 22)

[9] Hugo Touvron, Thibaut Lavril, Gautier Izacard, Xavier Martinet, Marie-Anne Lachaux, Timothée Lacroix, Baptiste Rozière, Naman Goyal, Eric Hambro, Faisal Azhar, Aurélien Rodriguez, Armand Joulin, Edouard Grave, and Guillaume Lample. Llama: Open and efficient foundation language models. *arXiv preprint*, arXiv:2302.13971, 2023. (Cited on pages 2 and 21)

[10] OpenAI. Gpt-4 technical report, 2023. (Cited on pages 2, 21 and 22)

[11] Emre Kiciman, Robert Ness, Amit Sharma, and Chenhao Tan. Causal reasoning and large language models: Opening a new frontier for causality. *arXiv preprint*, arXiv:2305.00050, 2023. (Cited on pages 2, 7, 8, 21 and 22)

[12] Sébastien Bubeck, Varun Chandrasekaran, Ronen Eldan, Johannes Gehrke, Eric Horvitz, Ece Kamar, Peter Lee, Yin Tat Lee, Yuanzhi Li, Scott M. Lundberg, Harsha Nori, Hamid Palangi, Marco Túlio Ribeiro, and Yi Zhang. Sparks of artificial general intelligence: Early experiments with GPT-4. *arXiv preprint*, arXiv:2303.12712, 2023. (Cited on pages 2, 21 and 22)

[13] Cheng Zhang, Stefan Bauer, Paul Bennett, Jiangfeng Gao, Wenbo Gong, Agrin Hilmkil, Joel Jennings, Chao Ma, Tom Minka, Nick Pawlowski, and James Vaughan. Understanding causality with large language models: Feasibility and opportunities. *arXiv preprint*, arXiv:2304.05524, 2023. (Cited on pages 2, 21 and 22)

[14] Ahmed Abdulaal, adamos hadjivasiliou, Nina Montana-Brown, Tiantian He, Ayodeji Ijishakin, Ivana Drobnjak, Daniel C. Castro, and Daniel C. Alexander. Causal modelling agents: Causal graph discovery through synergising metadata- and data-driven reasoning. In *The Twelfth International Conference on Learning Representations*, 2024. (Cited on pages 2, 21, 22 and 23)

[15] Matej Zečević, Moritz Willig, Devendra Singh Dhami, and Kristian Kersting. Causal parrots: Large language models may talk causality but are not causal. *Transactions on Machine Learning Research*, 2023. ISSN 2835-8856. (Cited on pages 2, 21 and 22)

[16] Zhijing Jin, Yuen Chen, Felix Leeb, Luigi Gresele, Ojasv Kamal, Zhiheng LYU, Kevin Blin, Fernando Gonzalez Adauto, Max Kleiman-Weiner, Mrinmaya Sachan, and Bernhard Schölkopf. CLadder: A benchmark to assess causal reasoning capabilities of language models. In *Thirty-seventh Conference on Neural Information Processing Systems*, 2023. (Cited on pages 2, 21 and 22)

[17] Zhijing Jin, Jiarui Liu, Zhiheng Lyu, Spencer Poff, Mrinmaya Sachan, Rada Mihalcea, Mona T. Diab, and Bernhard Schölkopf. Can large language models infer causation from correlation? *arXiv preprint*, arXiv:2306.05836, 2023. (Cited on pages 2, 21 and 22)

[18] Yue Zhang, Yafu Li, Leyang Cui, Deng Cai, Lemao Liu, Tingchen Fu, Xinting Huang, Enbo Zhao, Yu Zhang, Yulong Chen, Longyue Wang, Anh Tuan Luu, Wei Bi, Freda Shi, and Shuming Shi. Siren's song in the AI ocean: A survey on hallucination in large language models. *arXiv preprint*, arXiv:2309.01219, 2023. (Cited on pages 2, 21 and 22)

[19] Chenhang Cui, Yiyang Zhou, Xinyu Yang, Shirley Wu, Linjun Zhang, James Zou, and Huaxiu Yao. Holistic analysis of hallucination in gpt-4v(ision): Bias and interference challenges. *arXiv preprint*, arXiv:2311.03287, 2023. (Cited on pages 2, 21 and 22)

[20] Lukas Berglund, Meg Tong, Max Kaufmann, Mikita Balesni, Asa Cooper Stickland, Tomasz Korbak, and Owain Evans. The reversal curse: Llms trained on "a is b" fail to learn "b is a". *arXiv preprint*, arXiv:2309.12288, 2023. (Cited on pages 2, 21 and 22)

[21] Constantin F. Aliferis, Alexander Statnikov, Ioannis Tsamardinos, Subramani Mani, and Xenofon D. Koutsoukos. Local causal and markov blanket induction for causal discovery and feature selection for classification part i: Algorithms and empirical evaluation. *Journal of Machine Learning Research*, 11(7):171–234, 2010. (Cited on page 3)

[22] Shantanu Gupta, David Childers, and Zachary Chase Lipton. Local causal discovery for estimating causal effects. In *Conference on Causal Learning and Reasoning*, volume 213, pages 408–447, 2023. (Cited on page 3)

[23] Judea Pearl and Dana Mackenzie. *The Book of Why: The New Science of Cause and Effect*. Basic Books, Inc., USA, 1st edition, 2018. ISBN 046509760X. (Cited on pages 4 and 22)

[24] Timo Schick, Jane Dwivedi-Yu, Roberto Dessì, Roberta Raileanu, Maria Lomeli, Luke Zettlemoyer, Nicola Cancedda, and Thomas Scialom. Toolformer: Language models can teach themselves to use tools. *arXiv preprint*, arXiv:2302.04761, 2023. (Cited on page 4)

[25] Zhiheng Xi, Wenxiang Chen, Xin Guo, Wei He, Yiwen Ding, Boyang Hong, Ming Zhang, Junzhe Wang, Senjie Jin, Enyu Zhou, Rui Zheng, Xiaoran Fan, Xiao Wang, Limao Xiong, Yuhao Zhou, Weiran Wang, Changhao Jiang, Yicheng Zou, Xiangyang Liu, Zhangyue Yin, Shihan Dou, Rongxiang Weng, Wensen Cheng, Qi Zhang, Wenjuan Qin, Yongyan Zheng, Xipeng Qiu, Xuanjing Huan, and Tao Gui. The rise and potential of large language model based agents: A survey. *arXiv preprint*, arXiv:2309.07864, 2023. (Cited on page 4)

[26] Ruibo Tu, Kun Zhang, Bo C. Bertilson, Hedvig Kjellström, and Cheng Zhang. Neuropathic pain diagnosis simulator for causal discovery algorithm evaluation. In *Advances in Neural Information Processing Systems*, pages 12773–12784, 2019. (Cited on pages 4, 9, 10 and 43)

[27] Peter Spirtes, Christopher Meek, and Thomas Richardson. Causal inference in the presence of latent variables and selection bias. In *Uncertainty in Artificial Intelligence*, page 499–506, 1995. (Cited on pages 5, 7 and 22)

[28] Clark Glymour, Kun Zhang, and Peter Spirtes. Review of causal discovery methods based on graphical models. *Frontiers in Genetics*, 10, 2019. (Cited on pages 5 and 22)

[29] Jason Wei, Xuezhi Wang, Dale Schuurmans, Maarten Bosma, brian ichter, Fei Xia, Ed H. Chi, Quoc V Le, and Denny Zhou. Chain of thought prompting elicits reasoning in large language models. In Alice H. Oh, Alekh Agarwal, Danielle Belgrave, and Kyunghyun Cho, editors, *Advances in Neural Information Processing Systems*, 2022. (Cited on pages 5 and 22)

[30] Jiaxin Huang, Shixiang Gu, Le Hou, Yuexin Wu, Xuezhi Wang, Hongkun Yu, and Jiawei Han. Large language models can self-improve. In *Conference on Empirical Methods in Natural Language Processing*, pages 1051–1068, 2023. (Cited on page 5)

[31] Shuofei Qiao, Yixin Ou, Ningyu Zhang, Xiang Chen, Yunzhi Yao, Shumin Deng, Chuanqi Tan, Fei Huang, and Huajun Chen. Reasoning with language model prompting: A survey. In *Annual Meeting of the Association for Computational Linguistics (Volume 1: Long Papers)*, July 2023. (Cited on page 5)

[32] Judea Pearl. Probabilistic reasoning in intelligent systems - networks of plausible inference. In *Morgan Kaufmann series in representation and reasoning*, 1991. (Cited on page 5)

[33] Jonas Peters, Dominik Janzing, and Bernhard Schlkopf. *Elements of Causal Inference: Foundations and Learning Algorithms*. The MIT Press, 2017. ISBN 0262037319. (Cited on page 6)

[34] OpenAI. Hello, gpt-4o! https://openai.com/index/hello-gpt-4o/, 2024. Accessed: 2024-05-20. (Cited on page 7)

[35] Anthropic. Claude 3 family. https://www.anthropic.com/news/claude-3-family, 2024. Accessed: 2024-05-20. (Cited on page 7)

[36] Meta AI. Meta llama 3. https://ai.meta.com/blog/meta-llama-3/, 2024. Accessed: 2024-05-20. (Cited on page 7)

[37] Albert Q. Jiang, Alexandre Sablayrolles, Antoine Roux, Arthur Mensch, Blanche Savary, Chris Bamford, Devendra Singh Chaplot, Diego de Las Casas, Emma Bou Hanna, Florian Bressand, Gianna Lengyel, Guillaume Bour, Guillaume Lample, Lélio Renard Lavaud, Lucile Saulnier, Marie-Anne Lachaux, Pierre Stock, Sandeep Subramanian, Sophia Yang, Szymon Antoniak, Teven Le Scao, Théophile Gervet, Thibaut Lavril, Thomas Wang, Timothée Lacroix, and William El Sayed. Mixtral of experts. *arXiv preprint*, arXiv:2401.04088, 2024. (Cited on page 7)

[38] Sartaj Bhuvaji, Ankita Kadam, Prajakta Bhumkar, and Sameer Dedge. Brain tumor classification using deep learning algorithms. https://github.com/SartajBhuvaji/Brain-Tumor-Classification-Using-Deep-Learning-Algorithms, 2024. Accessed: 2024-05-19. (Cited on pages 9 and 43)

[39] Bidec Innovations. Stock price and news related to it, 2023. URL https://www.kaggle.com/datasets/BidecInnovations/stock-price-and-news-realted-to-it. Accessed: 2023-10-01. (Cited on pages 9 and 43)

[40] Gilbert P Compo, Jeffrey S Whitaker, Prashant D Sardeshmukh, Nobuki Matsui, Robert J Allan, Xungang Yin, Byron E Gleason, Russell S Vose, Glenn Rutledge, Pierre Bessemoulin, et al. The twentieth century reanalysis project. *Quarterly Journal of the Royal Meteorological Society*, 137(654):1–28, 2011. (Cited on pages 9, 10, 43 and 49)

[41] Jakob Bjerknes. Atmospheric teleconnections from the equatorial pacific. *Monthly weather review*, 97(3):163–172, 1969. (Cited on pages 10 and 51)

[42] Chunzai Wang. Enso, atlantic climate variability, and the walker and hadley circulations. In *The Hadley circulation: Present, past and future*, pages 173–202. Springer, 2004. (Cited on pages 10 and 51)

[43] Yinge Liu, Ninglian Wang, Lingang Wang, Zhongming Guo, and Xiaobo Wu. Variation of cloud amount over china and the relationship with enso from 1951 to 2014. *International Journal of Climatology*, 36(8):2931–2941, 2016. (Cited on pages 10 and 51)

[44] Anoop Kumar Mishra. Investigating changes in cloud cover using the long-term record of precipitation extremes. *Meteorological Applications*, 26(1):108–116, 2019. (Cited on pages 10 and 51)

[45] Peter Lee, Sebastien Bubeck, and Joseph Petro. Benefits, limits, and risks of gpt-4 as an ai chatbot for medicine. *New England Journal of Medicine*, 388(13):1233–1239, 2023. (Cited on page 21)

[46] Ruibo Tu, Chao Ma, and Cheng Zhang. Causal-discovery performance of chatgpt in the context of neuropathic pain diagnosis. *arXiv preprint*, arXiv:2301.13819, 2023. (Cited on pages 21 and 23)

[47] Judea Pearl and James M. Robins. Causal diagrams for epidemiologic research. *Epidemiology*, 10 1:37–48, 1999. (Cited on page 22)

[48] Shohei Shimizu, Patrik O Hoyer, Aapo Hyvärinen, Antti Kerminen, and Michael Jordan. A linear non-gaussian acyclic model for causal discovery. *Journal of Machine Learning Research*, 7(10), 2006. (Cited on page 22)

[49] Kun Zhang and Aapo Hyvarinen. On the identifiability of the post-nonlinear causal model. *arXiv preprint arXiv:1205.2599*, 2012. (Cited on page 22)

[50] Patrik Hoyer, Dominik Janzing, Joris M Mooij, Jonas Peters, and Bernhard Schölkopf. Nonlinear causal discovery with additive noise models. *Advances in neural information processing systems*, 2008. (Cited on page 22)

[51] Biwei Huang, Kun Zhang, Jiji Zhang, Joseph Ramsey, Ruben Sanchez-Romero, Clark Glymour, and Bernhard Schölkopf. Causal discovery from heterogeneous/nonstationary data. *Journal of Machine Learning Research*, 21(89):1–53, 2020. (Cited on pages 22 and 50)

[52] Karren Yang, Abigail Katcoff, and Caroline Uhler. Characterizing and learning equivalence classes of causal DAGs under interventions. In *International Conference on Machine Learning*, pages 5541–5550, 2018. (Cited on page 22)

[53] Philippe Brouillard, Sébastien Lachapelle, Alexandre Lacoste, Simon Lacoste-Julien, and Alexandre Drouin. Differentiable causal discovery from interventional data. In *Advances in Neural Information Processing Systems*, pages 21865–21877, 2020. (Cited on page 22)

[54] Joris M. Mooij, Sara Magliacane, and Tom Claassen. Joint causal inference from multiple contexts. *Journal of Machine Learning Research*, 21(99):1–108, 2020. (Cited on page 22)

[55] Ronan Perry, Julius von Kügelgen, and Bernhard Schölkopf. Causal discovery in heterogeneous environments under the sparse mechanism shift hypothesis. In *Advances in Neural Information Processing Systems*, pages 10904–10917, 2022. (Cited on page 22)

[56] Daniel Malinsky and Peter Spirtes. Learning the structure of a nonstationary vector autoregression. In *International Conference on Artificial Intelligence and Statistics*, pages 2986–2994, 2019. (Cited on page 22)

[57] Biwei Huang, Kun Zhang, Mingming Gong, and Clark Glymour. Causal discovery and forecasting in nonstationary environments with state-space models. In *International Conference on Machine Learning*, pages 2901–2910, 2019. (Cited on page 22)

[58] Chenxi Liu and Kun Kuang. Causal structure learning for latent intervened non-stationary data. In *International Conference on Machine Learning*, pages 21756–21777, 2023. (Cited on page 22)

[59] Zhengming Chen, Feng Xie, Jie Qiao, Zhifeng Hao, Kun Zhang, and Ruichu Cai. Identification of linear latent variable model with arbitrary distribution. In *AAAI Conference on Artificial Intelligence*, pages 6350–6357, 2022. (Cited on page 22)

[60] Anpeng Wu, Haoxuan Li, Kun Kuang, Zhang Keli, and Fei Wu. Learning causal relations from subsampled time series with two time-slices. In *International Conference on Machine Learning*, 2024. (Cited on page 22)

[61] Biwei Huang, Charles Jia Han Low, Feng Xie, Clark Glymour, and Kun Zhang. Latent hierarchical causal structure discovery with rank constraints. *Advances in neural information processing systems*, pages 5549–5561, 2022. (Cited on page 22)

[62] Xinshuai Dong, Biwei Huang, Ignavier Ng, Xiangchen Song, Yujia Zheng, Songyao Jin, Roberto Legaspi, Peter Spirtes, and Kun Zhang. A versatile causal discovery framework to allow causally-related hidden variables. *arXiv preprint*, arXiv:2312.11001, 2023. (Cited on page 22)

[63] Yibo Jiang and Bryon Aragam. Learning nonparametric latent causal graphs with unknown interventions. In *Advances in Neural Information Processing Systems*, volume 36, pages 60468–60513, 2023. (Cited on page 22)

[64] Xiu-Chuan Li, Kun Zhang, and Tongliang Liu. Causal structure recovery with latent variables under milder distributional and graphical assumptions. In *International Conference on Learning Representations*, 2024. (Cited on page 22)

[65] Aapo Hyvarinen, Hiroaki Sasaki, and Richard Turner. Nonlinear ica using auxiliary variables and generalized contrastive learning. In *International Conference on Artificial Intelligence and Statistics*, pages 859–868, 2019. (Cited on page 22)

[66] Ilyes Khemakhem, Diederik Kingma, Ricardo Monti, and Aapo Hyvarinen. Variational autoencoders and nonlinear ica: A unifying framework. In *International Conference on Artificial Intelligence and Statistics*, pages 2207–2217, 2020. (Cited on page 22)

[67] Francesco Locatello, Ben Poole, Gunnar Raetsch, Bernhard Schölkopf, Olivier Bachem, and Michael Tschannen. Weakly-supervised disentanglement without compromises. In *International Conference on Machine Learning*, pages 6348–6359, 2020. (Cited on page 22)

[68] David Klindt, Lukas Schott, Yash Sharma, Ivan Ustyuzhaninov, Wieland Brendel, Matthias Bethge, and Dylan Paiton. Towards nonlinear disentanglement in natural data with temporal sparse coding, 2021. (Cited on page 22)

[69] Sebastien Lachapelle, Pau Rodriguez, Yash Sharma, Katie E Everett, Rémi LE PRIOL, Alexandre Lacoste, and Simon Lacoste-Julien. Disentanglement via mechanism sparsity regularization: A new principle for nonlinear ICA. In *Conference on Causal Learning and Reasoning*, pages 428–484, 2022. (Cited on page 22)

[70] Kartik Ahuja, Jason S Hartford, and Yoshua Bengio. Weakly supervised representation learning with sparse perturbations. In *Advances in Neural Information Processing Systems*, pages 15516–15528, 2022. (Cited on page 22)

[71] Kartik Ahuja, Divyat Mahajan, Yixin Wang, and Yoshua Bengio. Interventional causal representation learning. In *International conference on machine learning*, pages 372–407, 2023. (Cited on page 22)

[72] Jiaqi Zhang, Kristjan Greenewald, Chandler Squires, Akash Srivastava, Karthikeyan Shanmugam, and Caroline Uhler. Identifiability guarantees for causal disentanglement from soft interventions. *Advances in Neural Information Processing Systems*, 2024. (Cited on page 22)

[73] Julius von Kügelgen, Michel Besserve, Liang Wendong, Luigi Gresele, Armin Kekić, Elias Bareinboim, David Blei, and Bernhard Schölkopf. Nonparametric identifiability of causal representations from unknown interventions. In *Advances in Neural Information Processing Systems*, pages 48603–48638, 2023. (Cited on page 22)

[74] Chandler Squires, Anna Seigal, Salil S Bhate, and Caroline Uhler. Linear causal disentanglement via interventions. In *International Conference on Machine Learning*, pages 32540–32560, 2023. (Cited on page 22)

[75] Simon Buchholz, Goutham Rajendran, Elan Rosenfeld, Bryon Aragam, Bernhard Schölkopf, and Pradeep Ravikumar. Learning linear causal representations from interventions under general nonlinear mixing. *Advances in Neural Information Processing Systems*, 2024. (Cited on page 22)

[76] Yongqiang Chen, Yonggang Zhang, Yatao Bian, Han Yang, Kaili Ma, Binghui Xie, Tongliang Liu, Bo Han, and James Cheng. Learning causally invariant representations for out-of-distribution generalization on graphs. In *Advances in Neural Information Processing Systems*, 2022. (Cited on page 22)

[77] Yongqiang Chen, Yatao Bian, Kaiwen Zhou, Binghui Xie, Bo Han, and James Cheng. Does invariant graph learning via environment augmentation learn invariance? In *Advances in Neural Information Processing Systems*, 2023. (Cited on page 22)

[78] Yongqiang Chen, Yatao Bian, Bo Han, and James Cheng. Interpretable and generalizable graph learning via subgraph multilinear extension. In *ICLR 2024 Workshop on Machine Learning for Genomics Explorations*, 2024. (Cited on page 22)

[79] Yongqiang Chen, Kaiwen Zhou, Yatao Bian, Binghui Xie, Kaili Ma, Yonggang Zhang, Han Yang, Bo Han, and James Cheng. Pareto invariant risk minimization. *arXiv preprint*, arXiv:2206.07766, 2022. (Cited on page 22)

[80] Yongqiang Chen, Wei Huang, Kaiwen Zhou, Yatao Bian, Bo Han, and James Cheng. Understanding and improving feature learning for out-of-distribution generalization. In *Advances in Neural Information Processing Systems*, 2023. (Cited on page 22)

[81] Long Ouyang, Jeff Wu, Xu Jiang, Diogo Almeida, Carroll L Wainwright, Pamela Mishkin, Chong Zhang, Sandhini Agarwal, Katarina Slama, Alex Ray, et al. Training language models to follow instructions with human feedback. *arXiv preprint arXiv:2203.02155*, 2022. (Cited on page 22)

[82] Karl Cobbe, Vineet Kosaraju, Mohammad Bavarian, Mark Chen, Heewoo Jun, Lukasz Kaiser, Matthias Plappert, Jerry Tworek, Jacob Hilton, Reiichiro Nakano, Christopher Hesse, and John Schulman. Training verifiers to solve math word problems. *arXiv preprint*, arXiv:2110.14168, 2021. (Cited on page 22)

[83] Nouha Dziri, Ximing Lu, Melanie Sclar, Xiang Lorraine Li, Liwei Jiang, Bill Yuchen Lin, Sean Welleck, Peter West, Chandra Bhagavatula, Ronan Le Bras, Jena D. Hwang, Soumya Sanyal, Xiang Ren, Allyson Ettinger, Zaid Harchaoui, and Yejin Choi. Faith and fate: Limits of transformers on compositionality. In *Thirty-seventh Conference on Neural Information Processing Systems*, 2023. (Cited on page 22)

[84] Bingbin Liu, Jordan T. Ash, Surbhi Goel, Akshay Krishnamurthy, and Cyril Zhang. Transformers learn shortcuts to automata. In *The Eleventh International Conference on Learning Representations*, 2023. (Cited on page 22)

[85] Xuan Li, Zhanke Zhou, Jianing Zhu, Jiangchao Yao, Tongliang Liu, and Bo Han. Deepinception: Hypnotize large language model to be jailbreaker, 2024. URL https://arxiv.org/abs/2311.03191. (Cited on page 22)

[86] Zhanke Zhou, Rong Tao, Jianing Zhu, Yiwen Luo, Zengmao Wang, and Bo Han. Can large language models reason robustly with noisy rationales? In *ICLR 2024 Workshop on Reliable and Responsible Foundation Models*, 2024. (Cited on page 22)

[87] Brett Drury, Hugo Gonçalo Oliveira, and Alneu de Andrade Lopes. A survey of the extraction and applications of causal relations. *Natural Language Engineering*, 28:361 – 400, 2022. (Cited on page 22)

[88] Shaobo Cui, Zhijing Jin, Bernhard Schölkopf, and Boi Faltings. The odyssey of common-sense causality: From foundational benchmarks to cutting-edge reasoning. *arXiv preprint arXiv:2406.19307*, 2024. (Cited on page 22)

[89] Pedram Hosseini, David A Broniatowski, and Mona Diab. Predicting directionality in causal relations in text. *arXiv preprint arXiv:2103.13606*, 2021. (Cited on page 22)

[90] Lei Gao, Prafulla Kumar Choubey, and Ruihong Huang. Modeling document-level causal structures for event causal relation identification. In *Proceedings of the 2019 Conference of the North American Chapter of the Association for Computational Linguistics: Human Language Technologies, Volume 1 (Long and Short Papers)*, pages 1808–1817, 2019. (Cited on page 22)

[91] Noah Weber, Rachel Rudinger, and Benjamin Van Durme. Causal inference of script knowledge. *arXiv preprint arXiv:2004.01174*, 2020. (Cited on page 22)

[92] Xiao Liu, Da Yin, Yansong Feng, Yuting Wu, and Dongyan Zhao. Everything has a cause: Leveraging causal inference in legal text analysis. *arXiv preprint arXiv:2104.09420*, 2021. (Cited on page 22)

[93] Andrew Kyle Lampinen, Stephanie C.Y. Chan, Ishita Dasgupta, Andrew Joo Hun Nam, and Jane X Wang. Passive learning of active causal strategies in agents and language models. In *Advances in Neural Information Processing Systems*, 2023. (Cited on page 22)

[94] Kristy Choi, Chris Cundy, Sanjari Srivastava, and Stefano Ermon. Lmpriors: Pre-trained language models as task-specific priors. *arXiv preprint*, arXiv:2210.12530, 2022. (Cited on pages 22 and 23)

[95] Stephanie Long, Alexandre Piché, Valentina Zantedeschi, Tibor Schuster, and Alexandre Drouin. Causal discovery with language models as imperfect experts. *arXiv preprint*, arXiv:2307.02390, 2023. (Cited on pages 22 and 23)

[96] Taiyu Ban, Lyuzhou Chen, Xiangyu Wang, and Huanhuan Chen. From query tools to causal architects: Harnessing large language models for advanced causal discovery from data. *arXiv preprint*, arXiv:2306.16902, 2023. (Cited on pages 22 and 23)

[97] Moritz Willig, Matej Zečević, Devendra Singh Dhami, and Kristian Kersting. Can foundation models talk causality? In *UAI 2022 Workshop on Causal Representation Learning*, 2022. (Cited on page 22)

[98] Stephanie Long, Tibor Schuster, and Alexandre Piché. Can large language models build causal graphs? *arXiv preprint*, arXiv:2303.05279, 2023. (Cited on page 22)

[99] Zhiheng LYU, Zhijing Jin, Rada Mihalcea, Mrinmaya Sachan, and Bernhard Schölkopf. Can large language models distinguish cause from effect? In *UAI 2022 Workshop on Causal Representation Learning*, 2022. (Cited on page 23)

[100] Yanming Zhang, Brette Fitzgibbon, Dino Garofolo, Akshith Kota, Eric Papenhausen, and Klaus Mueller. An explainable AI approach to large language model assisted causal model auditing and development. *arXiv preprint*, arXiv:2312.16211, 2023. (Cited on page 23)

[101] Lianmin Zheng, Wei-Lin Chiang, Ying Sheng, Siyuan Zhuang, Zhanghao Wu, Yonghao Zhuang, Zi Lin, Zhuohan Li, Dacheng Li, Eric Xing, et al. Judging llm-as-a-judge with mt-bench and chatbot arena. *Advances in Neural Information Processing Systems*, 36:46595–46623, 2023. (Cited on page 40)

[102] Daniel Lyndon, Joseph A Lansley, Jane Evanson, and Anant S Krishnan. Dural masses: meningiomas and their mimics. *Insights into imaging*, 10(1):11, 2019. (Cited on page 44)

[103] Nisreen Haydar, Khatoun Alyousef, Usama Alanan, Rana Issa, Fawaz Baddour, Zuheir Al-Shehabi, and Moatasem Hussein Al-Janabi. Role of magnetic resonance imaging (mri) in grading gliomas comparable with pathology: A cross-sectional study from syria. *Annals of Medicine and Surgery*, 82:104679, 2022. (Cited on page 44)

[104] Birkan Tunç, David A Hormuth II, George Biros, and Thomas E Yankeelov. Modeling of glioma growth with mass effect by longitudinal magnetic resonance imaging. *IEEE Transactions on Biomedical Engineering*, 68(12):3713–3724, 2021. (Cited on page 44)

[105] J Watts, G Box, A Galvin, P Brotchie, N Trost, and T Sutherland. Magnetic resonance imaging of meningiomas: a pictorial review. *Insights into imaging*, 5:113–122, 2014. (Cited on page 44)

[106] John Markoff. Competing as software goes to web. *The New York Times*, 2007. URL https://www.nytimes.com/2007/06/05/technology/05compute.html. Accessed: 2024-03-14. (Cited on page 47)

[107] Fama Eugene. Efficient capital markets: A review of theory and empirical work. *Journal of finance*, 25:383–417, 1970. (Cited on page 47)

[108] Jessica A Wachter. Can time-varying risk of rare disasters explain aggregate stock market volatility? *The Journal of Finance*, 68(3):987–1035, 2013. (Cited on page 47)

[109] Leland Bybee, Bryan T Kelly, Asaf Manela, and Dacheng Xiu. Business news and business cycles. *Journal of Finance, Forthcoming*, 2023. (Cited on page 47)

[110] Michael J McPhaden. Genesis and evolution of the 1997-98 el niño. *Science*, 283(5404): 950–954, 1999. (Cited on pages 49 and 50)

[111] Chester F Ropelewski and Michael S Halpert. Global and regional scale precipitation patterns associated with the el niño/southern oscillation. *Monthly weather review*, 115(8):1606–1626, 1987. (Cited on page 49)

[112] Wenju Cai, Simon Borlace, Matthieu Lengaigne, Peter Van Rensch, Mat Collins, Gabriel Vecchi, Axel Timmermann, Agus Santoso, Michael J McPhaden, Lixin Wu, et al. Increasing frequency of extreme el niño events due to greenhouse warming. *Nature climate change*, 4(2): 111–116, 2014. (Cited on page 49)

[113] Chunzai Wang, Clara Deser, Jin-Yi Yu, Pedro DiNezio, and Amy Clement. El niño and southern oscillation (enso): a review. *Coral reefs of the eastern tropical Pacific: Persistence and loss in a dynamic environment*, pages 85–106, 2017. (Cited on page 49)

[114] Joris M Mooij, Sara Magliacane, and Tom Claassen. Joint causal inference from multiple contexts. *Journal of Machine Learning Research*, 21(99):1–108, 2020. (Cited on page 50)

[115] Yi Ge Zhang, Mark Pagani, Jorijntje Henderiks, and Haojia Ren. A long history of equatorial deep-water upwelling in the pacific ocean. *Earth and Planetary Science Letters*, 467:1–9, 2017. ISSN 0012-821X. (Cited on page 50)

[116] J. Tarazona and W. Arntz. *The Peruvian Coastal Upwelling System*, pages 229–244. Springer Berlin Heidelberg, 2001. (Cited on page 51)

# Appendix of COAT

## Contents

# A Table of Notations

| Notation | Description |
|---|---|
| $\Psi$ | the LLM for factor proposal. |
| $\boldsymbol{p}^t$ | Prompt used at the $t$-th round for factor proposal. |
| $\Psi_p$ | the LLM for factor parsing. |
| $\boldsymbol{p}_p$ | Prompt used for factor parsing. |
| $\mathcal{A}$ | Algorithm for causal discovery. In this paper we mainly use FCI. |
| $\mathcal{F}$ | Abstraction of the COAT's Feedback procedure. |
| $\mathcal{D}$ | Full sample set. |
| $\widehat{\mathcal{D}}$ | A subset of sample set |
| $\widehat{\mathcal{D}}^t$ | The subset of sample used at the $t$-th round. |
| $\mathcal{C}$ | The predefined value space for LLM-proposed factors, e.g., set $\mathcal{C}$ as $\{-1, 0, 1\}$ |
| $\boldsymbol{x}$ | An unstructured sample of raw observation drawn from $\mathcal{X}$ |
| $y$ | A structured sample of target variable drawn from $\mathcal{Y}$ |
| $Y$ | target variable treated as a random variable. |
| $X$ | raw observation treated as a random variable. |
| $\boldsymbol{w}_i$ | The $i$-th factor proposed by LLMs, like *sweetness*, *size*, or *scent* |
| $\boldsymbol{w}_i(\cdot)$ | Factor's corresponding mapping from raw observation space $\mathcal{X}$ to the predefined value space $\mathcal{C}$. |
| $\mathcal{W}^t$ | The set of factors proposed by LLMs at the $t$-th round. |
| $\mathcal{W}^{\leq t}$ | The set of factors proposed by LLMs at the first $t$ rounds. |
| $\mathcal{S}$ | Index set of selected factors in $\mathcal{W}$, e.g., $\mathcal{S} = \{1, 3\}$ means only select the first and third factors. |
| $h_{\mathcal{S}}(\cdot)$ | LLM-induced representation over selection $\mathcal{S}$, e.g., $h_{\{1,3\}}(\cdot) = (\boldsymbol{w}_1(\cdot), \boldsymbol{w}_3(\cdot))$. $h(\cdot) := h_{[k]}(\cdot)$ |
| $\boldsymbol{Z}$ | Matrix of values of the structured high-level factors proposed by LLMs, i.e., $\widehat{\boldsymbol{Z}} \in \mathcal{C}^{n \times k}$ |
| $\boldsymbol{Z}^t$ | Matrix of values of the structured high-level factors proposed at the $t$-th COAT round. |
| $\boldsymbol{Z}^{\leq t}$ | Matrix of values of the structured high-level factors proposed at the first $t$ COAT rounds. |
| $\mathcal{G}^t$ | Causal graph at the $t$-th COAT round over $\widehat{\boldsymbol{Z}}^{\leq t} \cup \{Y\}$ |
| $k$ | Number of factors |
| $n$ | Number of samples |
| $C$ | Number of clusters |
| $C_\Psi$ | Capacity Score of LLM $\Psi$ |
| $p_\Psi$ or $p$ | Perception Score of LLM $\Psi$ |

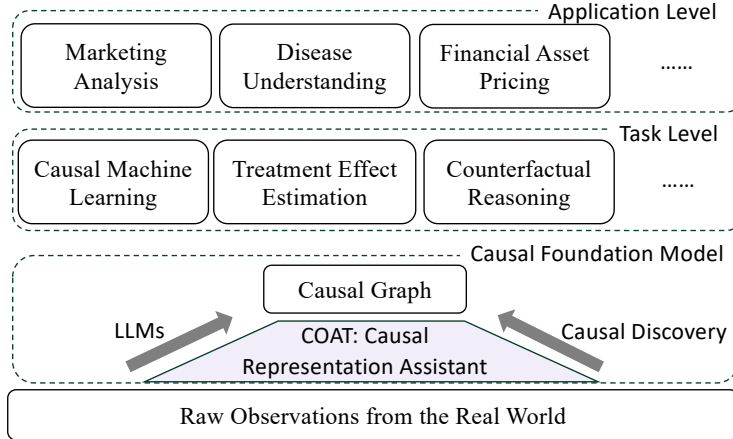

Figure 8: COAT combines both strengths of LLMs that learn the rich knowledge of the world, and causal discovery methods to uncover the hidden causal world. With the uncovered causal knowledge, COAT can empower broader applications of causal methods.

# B    Limitation and Future Opportunities

**Future Opportunities**   Despite the progress in the past decades, existing causal discovery algorithms mainly rely on high-quality measured variables given by human experts [3, 4, 5]. However, the causal variables and their measurements are usually available in a wide range of real-world applications. For example, Amazon sellers who want to analyze the factors related to user ratings only have user reviews, generated by the underlying user preferences for certain product characteristics. Therefore, the lack of measured high-quality causal variables has been a major impediment to broader real-world applications of causal or causality-inspired methods [6].

This work establishes and demonstrates a preliminary implementation of the system **C**ausal representati**O**n **A**ssistan**T** (COAT). We present comprehensive evaluations of COAT and find plentiful evidence that the recent emergence of Large Language Models (LLMs) [7, 8, 9, 10] has a great potential to mitigate the gap. In particular, as shown in Fig. 8, we envision an entangled system towards causal foundation models, which consists of two *mutually beneficial* components: LLMs and causal discovery methods (CDs). On the one hand, LLMs that learn rich world knowledge about the world can assist with discovering high-level hidden variables from low-level observational data [12], or even certain commonsense causal knowledge [11, 13, 14]. On the other hand, it has also been found that the reliability of LLMs in the reasoning of causality remains a debate [13, 15, 16, 17], due to a series of drawbacks of LLMs [18, 19, 20], as well as the ethical considerations [45, 46]. Nevertheless, CDs can uncover the underlying causal relations between the discovered high-level variables with guarantees. More importantly, CDs can also provide feedback to improve the identification of the variables. The combination of LLMs and CDs iteratively improves the discovery of the hidden causal world, and hence opens up a broader adoption of various causal methods.

**Limitations**   We would like to discuss some technical limitations of the current version of COAT.

- **Multicollinearity.**   Sometimes, LLM may output factors with shared similar semantic meanings or overlapped factor-value criteria. The consequence of this phenomenon is that multicollinearity could occur in the resulting structured data, or the data matrix (each sample per row) would not be full column rank. This would hinder the conditional independent tests. In experiments, we adopt numerical methods like PCA to drop similar factors. More sophisticated methods could be investigated in the future.

- **Toward complete causal graph.**   In this work, we mainly focus on one single target variable and identify a set of factors to serve as a Markov blanket of this target variable. Indeed, one can explore a more complete causal graph by applying COAT procedures on those identified factors to acquire a more comprehensive causal graph. One may also introduce multiple target variables or let LLM define a suitable target variable (and also parse it out) from pure raw observations. These extensions are also important, and we leave it to future work.

Meanwhile, identifiable causal discovery also requires certain assumptions about the data:

- **Faithfulness.** The empirical distribution of the data reflects the actual data-generating process.
- **No selection bias.** Otherwise, the condition of faithfulness would be violated.
- **Enough sample size.** Our method involves statistical tests; the more, the better.

## C Related Work

**Causal discovery** aims to discover the *unknown* causal relations from the observational data [3, 4], which is critical to both real-world applications and scientific discoveries [23, 47]. We use FCI [27] in this paper just to illustrate the idea. To attain identifiability better than the Markov equivalent class, one can choose more advanced Causal Discovery methods under different assumptions, like constrained functional [48, 49, 50], multiple domain data [51, 52, 53, 54, 55], and non-stationary data [51, 56, 57, 58]. There are also some works aiming to address realistic challenges, like structure models with arbitrary distribution [59], subsampled time series [60], and latent hierarchical causal structure [61, 62, 63, 64]. Despite recent theoretical and empirical improvements [5, 28], most existing causal discovery approaches rely on well-structured data with human-crafted factors as inputs. They can easily suffer from the low quality of annotated data (e.g., latent confounders).

**Causal representation learning** aims to establish provable identifiability for latent high-level variables (like location and color of an object) from low-level observations such as images [6]. Such identification can be feasible with certain conditions, like auxiliary information [65, 66, 67], sparsity [68, 69, 70], or interventional data [71, 72, 73, 74, 75]. Recent works also generalize the causal representation learning to graph-structured data [76, 77, 78], and discuss the optimization challenges in realizing causal representation learning [79, 80]. In this work, we show that incorporating LLMs that learn world knowledge from massive training data could effectively relieve the need for artificial causal factor annotation. Meanwhile, COAT also opens up a new line to learn causal representations with rich pre-trained knowledge as well as feedback from causal discovery algorithms.

**Reasoning with LLMs** has achieved remarkable performance across a variety of tasks with few demonstrations of the samples [7, 8, 10, 81]. The strong capabilities of LLMs show that it is evident that LLMs could acquire and understand commonsense knowledge about the world [12]. The power of LLMs can be further unlocked with suitable context as inputs [29]. Nevertheless, LLMs have also been found to make mistakes in basic algorithmic reasoning [82, 83], easy to hallucinate nonfactual results [18], tend to learn shortcuts or dataset biases [19, 20, 84], vulnerable to adversarial jailbreaks or noisy interruptions [85, 86], and fall short in complex planning and reasoning tasks [12]. The drawbacks of LLMs render it *risky* to rely on the direct LLM reasoning results to derive any rigorous results. Therefore, we do not directly derive the results from LLMs. Rather, we merely leverage the learned world knowledge in LLMs to find useful causal factors by constructing proper instructions based on causal feedback.

**Text mining of causal relations** aims to analyze the causal relations implied by the semantics given a text [87, 88, 89]. In particular, some works focus on the identification of causal relations among events specified in documents [90, 91, 92]. Different from these tasks, where factors are entities or events described by text, in this work, COAT emphasizes crafting high-level factors that go beyond the unstructured data like text description and also establish identifiability on the Markov blanket for a given target variable. Note that the target variable is allowed to be not mentioned in the text or other unstructured data. Therefore, this work brings new scope and opportunities for these text-understanding tasks.

**Causal learning with LLMs** has received muchattention results from the community [11, 13]. Kiciman et al. [11] find that LLMs can recover the pairwise causal relations very well. Lampinen et al. [93] show that transformer-based agents can learn causal strategies passively if intervention is allowed during tests. Abdulaal et al. [14], Choi et al. [94], Long et al. [95], Ban et al. [96] propose to incorporate the causal discovery results by LLMs as a prior or constraint to improve the performance of data-driven causal discovery algorithms. However, Zečević et al. [15], Willig et al. [97] find that LLMs can not understand causality but simply retell the causal knowledge contained in the training data. Zhang et al. [13], Jin et al. [16, 17] find that LLMs can hardly provide satisfactory answers for discovering new knowledge or decision-making tasks. Although Long et al. [98] find that LLMs can

build causal graphs with 3-4 nodes, Tu et al. [46] find that the performance of LLMs in more complex causal discovery remains limited as LLMs can hardly understand new concepts and knowledge. The aforementioned debate implies the limitations in directly adopting the causal discovery results by LLMs, which motivates us to incorporate the existing causal discovery algorithms with rigorous guarantees instead of LLMs to learn the causal relations.

LYU et al. [99] find that it is crucial to incorporate prompts aligned with the underlying causal story for LLMs to do pairwise causal relation inference. Zhang et al. [100] propose to leverage LLMs to audit causal discovery results in an explainable way.

The closest works to ours are Abdulaal et al. [14], Choi et al. [94], Long et al. [95], Ban et al. [96] which also incorporates LLMs into the pipeline of causal discovery. Nevertheless, all of the existing combinations of LLMs and causal discovery still focus on artificially curated structured data and rely on the capability of LLMs to infer causal relations, therefore, limited in both the reliability and the utility of LLMs in causal learning.

# D  Proofs for Theoretical Results

## D.1  Proof for Proposition 2.2

**Proposition D.1** (Restatement of Proposition 2.2). *Given the current representation as $h_{[k]}(X) = (\boldsymbol{w}_1(X), \boldsymbol{w}_2(X), \cdots, \boldsymbol{w}_k(X))$, and a new factor $\boldsymbol{w}_{k+1}$ satisfying*

$$Y \not\!\perp\!\!\!\perp \boldsymbol{w}_{k+1}(X) \mid h_{[k]}(X), \tag{12}$$

*for Markov Blanket $\mathcal{S} \subseteq [k+1]$ of $Y$, i.e.,*

$$Y \perp\!\!\!\perp h_{[k+1]\setminus\mathcal{S}}(X) \mid h_{\mathcal{S}}(X), \tag{13}$$

*we have the following result about conditional mutual information:*

$$I(Y; X \mid h_{\mathcal{S}}(X)) = I\left(Y; X \mid h_{[k+1]}(X)\right) < I\left(Y; X \mid h_{[k]}(X)\right) \tag{14}$$

*Proof for Prop. 2.2.* From Eq. 12, we have

$$H(y \mid h_{[k]}(\boldsymbol{x}); \boldsymbol{w}_{k+1}(\boldsymbol{x})) < H(y \mid h_{[k]}(\boldsymbol{x})); \tag{15}$$

From Eq. 13, we have

$$H(y \mid h_{\mathcal{S}}(\boldsymbol{x}); h_{[k+1]\setminus\mathcal{S}}(\boldsymbol{x})) = H(y \mid h_{\mathcal{S}}(\boldsymbol{x})). \tag{16}$$

Therefore:

$$\begin{aligned}
H(y \mid h_{\mathcal{S}}(\boldsymbol{x})) &= H(y \mid h_{\mathcal{S}}(\boldsymbol{x}), h_{[k+1]\setminus\mathcal{S}}(\boldsymbol{x})) \\
&= H(y \mid h_{[k+1]}(\boldsymbol{x})) \\
&= H(y \mid h_{[k]}(\boldsymbol{x}), \boldsymbol{w}_{k+1}(\boldsymbol{x})) \\
&< H(y \mid h_{[k]}(\boldsymbol{x}))
\end{aligned} \tag{17}$$

Also note that

$$H(y \mid h_{\mathcal{S}}(\boldsymbol{x}), \boldsymbol{x}) = H(y \mid \boldsymbol{x}) = H(y \mid h_{[k]}(\boldsymbol{x}), \boldsymbol{x}), \tag{18}$$

therefore:

$$H(y \mid h_{\mathcal{S}}(\boldsymbol{x})) - H(y \mid h_{\mathcal{S}}(\boldsymbol{x}), \boldsymbol{x}) < H(y \mid h_{[k]}(\boldsymbol{x})) - H(y \mid h_{[k]}(\boldsymbol{x}), \boldsymbol{x}), \tag{19}$$

which is Eq. 14. □

## D.2  Proof for Proposition 2.4

**Proposition D.2** (Restatement of Proposition 2.4). *Given Assumption 2.3, for any small number $\epsilon, \delta \in (0, \frac{1}{2})$, with sufficiently $t$ rounds of COAT, i.e.,*

$$\sqrt{t} > \frac{|z_\delta|\sqrt{1-p}}{2\sqrt{p}} \left(1 + \sqrt{1 + \frac{4\log\epsilon}{z_\delta^2(1-p)\log 1 - C_\Psi}}\right), \tag{20}$$

*where $z_\delta$ is the $\delta$-quantile of the standard normal distribution, we have*

$$\Pr\left(\frac{I(Y; X \mid h_{\leq t}(X))}{I(Y; X)} < \epsilon\right) \geq 1 - \delta. \tag{21}$$

*Proof for Prop. 2.4.* Let $n_s$ be the number of rounds that LLM proposed at least one usable factor satisfying Assumption 2.3. Since $t$ is large, its Binomial distribution $\mathrm{Binom}(t, p)$ can be approximated by Gaussian distribution $\mathcal{N}(tp, tp(1-p))$. To enforce

$$
\begin{aligned}
\Pr\left(\frac{I\left(y; \boldsymbol{x} \mid h_{\leq t}(\boldsymbol{x})\right)}{I\left(y; \boldsymbol{x}\right)} < \epsilon\right) &\geq \Pr\left((1 - C_\Psi)^{n_s} < \epsilon\right) \\
&= \Pr\left(n_s > \frac{\log \epsilon}{\log\left(1 - C_\Psi\right)}\right) \\
&= \Pr\left(\frac{n_s - tp}{\sqrt{tp(1-p)}} > \frac{1}{\sqrt{tp(1-p)}}\left(\frac{\log \epsilon}{\log\left(1 - C_\Psi\right)} - tp\right)\right) \\
&\geq 1 - \delta,
\end{aligned}
\tag{22}
$$

we have

$$
\frac{1}{\sqrt{tp(1-p)}}\left(\frac{\log \epsilon}{\log\left(1 - C_\Psi\right)} - tp\right) < z_\delta = -|z_\delta|
\tag{23}
$$

with Gaussian distribution approximation.

Isolating $\sqrt{t} > 0$ from the above inequality, we would have the desired result. $\qquad\square$

Table 3: Causal relation extraction results in AppleGastronome.

| LLM | METHOD | SHD | SID | RECALL | PRECISION | F1 |
|---|---|---|---|---|---|---|
| GPT 4O | PAIRWISE | 5.33±2.62 | 0.00±0.00 | 1.00±0.00 | 0.45±0.07 | 0.62±0.07 |
|  | COAT | 1.00±0.00 | 0.00±0.00 | 1.00±0.00 | 0.79±0.03 | 0.89±0.02 |
| GPT 4 | PAIRWISE | 8.67±2.05 | 1.33±1.89 | 0.93±0.09 | 0.35±0.05 | 0.51±0.06 |
|  | COAT | 2.33±0.47 | 1.00±1.41 | 0.93±0.09 | 0.70±0.09 | 0.79±0.04 |
| GPT 3.5 | PAIRWISE | 5.67±4.03 | 3.00±2.94 | 0.67±0.24 | 0.49±0.36 | 0.55±0.32 |
|  | COAT | 2.00±0.82 | 2.67±2.05 | 0.75±0.20 | 0.63±0.21 | 0.68±0.21 |
| MISTRAL-LARGE | PAIRWISE | 7.33±1.89 | 0.00±0.00 | 1.00±0.00 | 0.38±0.03 | 0.55±0.03 |
|  | COAT | 2.00±0.00 | 1.33±1.25 | 0.85±0.11 | 0.74±0.05 | 0.78±0.02 |
| MISTRAL-MEDIUM | PAIRWISE | 6.00±1.63 | 2.00±0.00 | 0.51±0.13 | 0.37±0.04 | 0.42±0.07 |
|  | COAT | 0.33±0.47 | 0.00±0.00 | 1.00±0.00 | 0.94±0.08 | 0.97±0.04 |
| LLAMA-3-70B | PAIRWISE | 4.00±1.41 | 0.33±0.47 | 0.92±0.12 | 0.51±0.09 | 0.65±0.11 |
|  | COAT | 2.33±0.47 | 3.67±1.25 | 0.75±0.00 | 0.70±0.07 | 0.72±0.04 |
| LLAMA-2-70B | PAIRWISE | 3.33±1.25 | 1.00±1.41 | 0.89±0.16 | 0.48±0.11 | 0.62±0.13 |
|  | COAT | 1.00±0.82 | 0.33±0.47 | 0.89±0.16 | 0.81±0.14 | 0.84±0.14 |
| QWEN-1.5-110B | PAIRWISE | 6.67±2.62 | 2.33±1.25 | 0.62±0.03 | 0.38±0.09 | 0.47±0.07 |
|  | COAT | 4.00±1.63 | 7.00±4.32 | 0.47±0.34 | 0.47±0.34 | 0.70±0.10 |
| DEEPSEEK-V2 | PAIRWISE | 5.00±1.41 | 1.33±1.89 | 0.93±0.09 | 0.47±0.05 | 0.62±0.06 |
|  | COAT | 1.33±1.25 | 2.33±1.70 | 0.85±0.11 | 0.87±0.19 | 0.85±0.14 |
| CLAUDE-3-OPUS | PAIRWISE | 7.33±1.25 | 0.33±0.47 | 0.93±0.09 | 0.40±0.05 | 0.56±0.07 |
|  | COAT | 1.33±0.47 | 0.00±0.00 | 1.00±0.00 | 0.79±0.06 | 0.88±0.04 |

Table 4: Independence tests of the annotation noises with annotated features and other noises AppleGastronome.

| LLM | TEST OBJECT | T | P-VALUE |
|---|---|---|---|
| GPT 4 | FEATURE | 0.2828 | 0.9997 |
|  | NOISE | 0.0327 | 0.9962 |
| GPT 3.5 | FEATURE | 0.4803 | 0.0325 |
|  | NOISE | 0.0446 | 0.9962 |

# E More Details about Experiments

## E.1 More Details on Constructing AppleGastronome

In the AppleGastronome benchmark, we consider the target variable as a rating score of the apple by several gastronomes. Each apple has its own attributes, including size, smell, and taste (or sweetness). Each gastronome has a unique preference for some attributes of the apple. They will give and rating as well as write a review according to the matchness of the apple with respect to their preference. We generate the review using GPT 4 by fetching GPT 4 the preferences and the apple attributes.

The prompts for generating the unstructured inputs are given in Fig. 9. The additional results on Relation Extraction are given in Table. 3.

Examples of AppleGastronome are given in Fig. 10.

## E.2 More Details on Prompts for AppleGastronome

The prompts for factor proposal are given in Fig. 11.

The prompt for factor annotation is given in Fig. 12.

```
You are a picky gastronome on apples. You
are ready to evaluate apples and write
reviews. Your writing should be clear,
solid and convincing to suppliers and
customers.

Task
Please write a short review about the
evaluation results for a given apple.

Evaluation Results:
- a high content of essential nutrients,
including dietary fiber, vitamin C, etc.
- musty or rotten
- more sour than sweet
- dry and lacking Moisture
- might not bring the expected returns and
could even lead to losses

Additional Requirement:
- Combine those evaluation results into
more detailed reasoning.
- Single paragraph; No quotation marks.
- Modern English
- Begin this part by a mark [Review Begin].
- no more than 60 words.
```

Figure 9: Illustration of prompts for generating AppleGastronome.

```
## Score: -3
- The assessed apple falls short in offering the essential nutrients that consumers demand,
presenting a concerning deficiency. Its diminutive stature coupled with a distinct tartness that
overshadows any sweetness detracts from its palate appeal. Moreover, the dry texture fails to
satisfy those seeking a juicy bite. Given these drawbacks, this apple variety may disappoint
buyers, posing a financial risk for suppliers.

- Despite its diminutive stature, this apple fails to meet even basic quality expectations. The
  presence of a musty or rotten scent signals underlying issues that can neither be overlooked
  by discerning consumers nor excused by vendors seeking profit. Ultimately, stocking this fruit
  is a risky venture that's far more likely to incur losses than yield any anticipated returns.

… …

## Score: 3
- This apple variety, rich in essential nutrients like dietary fiber and vitamin C, impresses
with its large size and delightful strong aroma. Its balanced flavor, leaning more towards sweet
than sour, and abundant refreshing moisture guarantee a delectable eating experience. This
apple's significant market potential is evident, and it certainly merits broader acclaim for its
high-quality attributes.
```

Figure 10: Illustration of examples in AppleGastronome.

The prompts for constructing feedback are given in Fig. 13.

### E.3   More Details of Results on AppleGastronome

The detailed causal graph results are given from Fig. 14 to Fig. 18.

Independent tests about annotation on the Apple Gastronome benchmark are shown in Table 4.

```
You are an excellently helpful AI
assistant for analyzing, abstracting,
and processing data. Now try your
best.

Rating=0
…
Rating=1
…
Rating=2
…

Tasks: Factor Abstraction.
What are the high-level factors behind
the text that contribute to the
allocation of groups?
• Propose as many as possible factors
  based on your knowledge.
• Adjust your candidate factors by
  your observation on given data.
• The proposed factors should be
  diverse and different semantically.
  Their overlapping should be
  minimized.
```

Figure 11: Illustration of the prompt for factor proposal.

```
You are an excellently helpful AI assistant
for analyzing, abstracting, and processing
data.
Now try your best.

# Data

The reviewer comments on an apple that are
randomly picked:
{text}

# Tasks

For the given sample, what is the most
appreciate factor value based on the
criterion?

{factor_state}

**Your final output should follow this
template**:
**"The value is: ___."**
```

Figure 12: Illustration of the prompt for factor annotation.

```
# Data
… <groups of samples> …

# Tasks: Factor Abstraction.

**What are the high-level factors behind the text that contribute to the
allocation of groups?**

- Propose your candidate factors based on your observation on given data.
- The proposed factors should be diverse and different semantically. Their
  overlapping should be minimized.

# About Output

Your output should contain parts described as follows.

**Part 1**: Consideration.

In this part, feel free to write down the process of your considerations.
Hint: You need to abstract, identify, and design suitable factors with
corresponding criteria, and each factor is only allowed to take value from
[-1, 0, 1] .

**Part 2**: Factor filtration. You should decide whether to use each of the
proposed factors by following criteria:

- Each new factor should be helpful to distinguish the groups.
- Each Factor should focus on one concrete aspect and try to avoid
  overlapping.
- Avoid overlapping with those existing factors:
  - size
  - aroma
```

Figure 13: Illustration of the prompt for feedback.

## E.4 The Full Result on the Apple Gastronome Benchmark

Full Results on the Apple Gastronome Benchmark are shown in Table 5.

Full Results of Causal Metrics each Round of each LLM on Apple Gastronome Benchmark are shown in Table 6.

Table 5: Full Results on the Apple Gastronome Benchmark.

| LLMs | METHOD | MB | NMB | OT | RECALL | PRECITION | F1 |
|---|---|---|---|---|---|---|---|
| GPT 4O | META | $3.00_{\pm0.82}$ | $1.00_{\pm0.00}$ | $4.00_{\pm0.82}$ | $0.60_{\pm0.16}$ | $0.37_{\pm0.09}$ | $0.46_{\pm0.12}$ |
| | DATA | $3.00_{\pm0.00}$ | $0.67_{\pm0.47}$ | $0.00_{\pm0.00}$ | $0.60_{\pm0.00}$ | $0.83_{\pm0.12}$ | $0.69_{\pm0.04}$ |
| | DATA+CoT | $4.67_{\pm0.58}$ | $1.00_{\pm0.00}$ | $0.33_{\pm0.58}$ | $0.93_{\pm0.12}$ | $0.78_{\pm0.10}$ | $0.85_{\pm0.10}$ |
| | COAT | $4.00_{\pm0.82}$ | $0.00_{\pm0.00}$ | $0.00_{\pm0.00}$ | $0.80_{\pm0.16}$ | $1.00_{\pm0.00}$ | $0.88_{\pm0.10}$ |
| GPT 4 | META | $2.67_{\pm0.94}$ | $0.67_{\pm0.47}$ | $2.33_{\pm0.47}$ | $0.53_{\pm0.19}$ | $0.46_{\pm0.08}$ | $0.49_{\pm0.13}$ |
| | DATA | $3.00_{\pm0.00}$ | $0.33_{\pm0.47}$ | $0.00_{\pm0.00}$ | $0.60_{\pm0.00}$ | $0.92_{\pm0.12}$ | $0.72_{\pm0.04}$ |
| | DATA+CoT | $4.33_{\pm0.58}$ | $0.83_{\pm0.29}$ | $0.17_{\pm0.29}$ | $0.87_{\pm0.12}$ | $0.81_{\pm0.02}$ | $0.84_{\pm0.06}$ |
| | COAT | $4.00_{\pm0.82}$ | $0.33_{\pm0.47}$ | $0.00_{\pm0.00}$ | $0.80_{\pm0.16}$ | $0.93_{\pm0.09}$ | $0.85_{\pm0.11}$ |
| GPT 3.5 | META | $3.33_{\pm1.25}$ | $0.33_{\pm0.47}$ | $4.33_{\pm1.25}$ | $0.67_{\pm0.25}$ | $0.42_{\pm0.12}$ | $0.51_{\pm0.15}$ |
| | DATA | $2.67_{\pm0.47}$ | $0.67_{\pm0.47}$ | $0.00_{\pm0.00}$ | $0.53_{\pm0.09}$ | $0.81_{\pm0.14}$ | $0.64_{\pm0.10}$ |
| | DATA+CoT | $5.00_{\pm0.00}$ | $1.00_{\pm0.00}$ | $1.33_{\pm0.58}$ | $1.00_{\pm0.00}$ | $0.68_{\pm0.05}$ | $0.81_{\pm0.04}$ |
| | COAT | $3.67_{\pm0.47}$ | $0.00_{\pm0.00}$ | $0.00_{\pm0.00}$ | $0.73_{\pm0.09}$ | $1.00_{\pm0.00}$ | $0.84_{\pm0.07}$ |
| MISTRAL-LARGE | META | $2.00_{\pm0.82}$ | $0.67_{\pm0.47}$ | $2.67_{\pm0.94}$ | $0.40_{\pm0.16}$ | $0.37_{\pm0.09}$ | $0.38_{\pm0.12}$ |
| | DATA | $3.00_{\pm0.00}$ | $0.33_{\pm0.47}$ | $0.00_{\pm0.00}$ | $0.60_{\pm0.00}$ | $0.92_{\pm0.12}$ | $0.72_{\pm0.04}$ |
| | DATA+CoT | $4.33_{\pm0.58}$ | $1.00_{\pm0.00}$ | $0.67_{\pm0.58}$ | $0.87_{\pm0.12}$ | $0.73_{\pm0.07}$ | $0.79_{\pm0.05}$ |
| | COAT | $4.33_{\pm0.47}$ | $0.00_{\pm0.00}$ | $0.00_{\pm0.00}$ | $0.87_{\pm0.09}$ | $1.00_{\pm0.00}$ | $0.93_{\pm0.05}$ |
| MISTRAL-MEDIUM | META | $3.00_{\pm0.00}$ | $0.67_{\pm0.47}$ | $1.67_{\pm1.25}$ | $0.60_{\pm0.00}$ | $0.59_{\pm0.13}$ | $0.59_{\pm0.07}$ |
| | DATA | $3.00_{\pm0.00}$ | $0.67_{\pm0.47}$ | $0.00_{\pm0.00}$ | $0.60_{\pm0.00}$ | $0.83_{\pm0.12}$ | $0.69_{\pm0.04}$ |
| | DATA+CoT | $4.33_{\pm0.58}$ | $1.00_{\pm0.00}$ | $0.67_{\pm0.58}$ | $0.87_{\pm0.12}$ | $0.73_{\pm0.07}$ | $0.79_{\pm0.05}$ |
| | COAT | $4.67_{\pm0.47}$ | $0.00_{\pm0.00}$ | $0.00_{\pm0.00}$ | $0.93_{\pm0.09}$ | $1.00_{\pm0.00}$ | $0.96_{\pm0.05}$ |
| LLAMA-3-70B | META | $2.67_{\pm0.47}$ | $0.33_{\pm0.47}$ | $4.67_{\pm0.47}$ | $0.53_{\pm0.09}$ | $0.35_{\pm0.06}$ | $0.42_{\pm0.07}$ |
| | DATA | $2.67_{\pm1.25}$ | $0.33_{\pm0.47}$ | $0.00_{\pm0.00}$ | $0.53_{\pm0.25}$ | $0.93_{\pm0.09}$ | $0.63_{\pm0.21}$ |
| | DATA+CoT | $2.67_{\pm0.58}$ | $0.67_{\pm0.58}$ | $1.00_{\pm0.00}$ | $0.53_{\pm0.12}$ | $0.62_{\pm0.13}$ | $0.57_{\pm0.11}$ |
| | COAT | $3.67_{\pm0.47}$ | $0.33_{\pm0.47}$ | $0.00_{\pm0.00}$ | $0.73_{\pm0.09}$ | $0.93_{\pm0.09}$ | $0.81_{\pm0.06}$ |
| LLAMA-2-70B | META | $2.33_{\pm0.47}$ | $0.67_{\pm0.47}$ | $4.67_{\pm1.25}$ | $0.47_{\pm0.09}$ | $0.32_{\pm0.07}$ | $0.37_{\pm0.06}$ |
| | DATA | $2.33_{\pm0.94}$ | $0.67_{\pm0.47}$ | $0.00_{\pm0.00}$ | $0.47_{\pm0.19}$ | $0.75_{\pm0.20}$ | $0.57_{\pm0.20}$ |
| | DATA+CoT | $3.00_{\pm1.73}$ | $0.67_{\pm0.58}$ | $0.33_{\pm0.58}$ | $0.60_{\pm0.35}$ | $0.71_{\pm0.34}$ | $0.65_{\pm0.35}$ |
| | COAT | $3.00_{\pm0.00}$ | $0.67_{\pm0.47}$ | $0.00_{\pm0.00}$ | $0.60_{\pm0.00}$ | $0.83_{\pm0.12}$ | $0.69_{\pm0.04}$ |
| QWEN-1.5-110B | META | $2.00_{\pm0.00}$ | $1.00_{\pm0.00}$ | $4.00_{\pm0.00}$ | $0.40_{\pm0.00}$ | $0.29_{\pm0.00}$ | $0.33_{\pm0.00}$ |
| | DATA | $3.00_{\pm0.82}$ | $1.00_{\pm0.00}$ | $0.00_{\pm0.00}$ | $0.60_{\pm0.16}$ | $0.74_{\pm0.05}$ | $0.66_{\pm0.12}$ |
| | DATA+CoT | $3.67_{\pm1.53}$ | $0.67_{\pm0.58}$ | $0.00_{\pm0.00}$ | $0.73_{\pm0.31}$ | $0.83_{\pm0.17}$ | $0.77_{\pm0.23}$ |
| | COAT | $4.00_{\pm0.82}$ | $0.33_{\pm0.47}$ | $0.00_{\pm0.00}$ | $0.80_{\pm0.16}$ | $0.93_{\pm0.09}$ | $0.85_{\pm0.11}$ |
| DEEPSEEK-V2 | META | $2.33_{\pm0.47}$ | $1.00_{\pm0.00}$ | $3.00_{\pm0.82}$ | $0.47_{\pm0.09}$ | $0.37_{\pm0.03}$ | $0.41_{\pm0.04}$ |
| | DATA | $3.33_{\pm0.47}$ | $0.67_{\pm0.47}$ | $0.00_{\pm0.00}$ | $0.67_{\pm0.09}$ | $0.85_{\pm0.11}$ | $0.74_{\pm0.05}$ |
| | DATA+CoT[1] | $5.00_{\pm0.00}$ | $1.00_{\pm0.00}$ | $0.67_{\pm0.58}$ | $1.00_{\pm0.00}$ | $0.75_{\pm0.07}$ | $0.86_{\pm0.04}$ |
| | COAT | $3.67_{\pm0.47}$ | $0.67_{\pm0.47}$ | $0.00_{\pm0.00}$ | $0.73_{\pm0.09}$ | $0.87_{\pm0.09}$ | $0.78_{\pm0.02}$ |
| CLAUDE-3-OPUS | META | $2.00_{\pm0.00}$ | $1.00_{\pm0.00}$ | $3.00_{\pm1.41}$ | $0.40_{\pm0.00}$ | $0.35_{\pm0.07}$ | $0.37_{\pm0.04}$ |
| | DATA | $3.33_{\pm0.47}$ | $0.33_{\pm0.47}$ | $0.00_{\pm0.00}$ | $0.67_{\pm0.09}$ | $0.93_{\pm0.09}$ | $0.77_{\pm0.02}$ |
| | DATA+CoT | $2.67_{\pm0.58}$ | $0.33_{\pm0.58}$ | $1.00_{\pm1.00}$ | $0.53_{\pm0.12}$ | $0.67_{\pm0.14}$ | $0.59_{\pm0.13}$ |
| | COAT | $5.00_{\pm0.00}$ | $0.33_{\pm0.47}$ | $0.00_{\pm0.00}$ | $1.00_{\pm0.00}$ | $0.94_{\pm0.08}$ | $0.97_{\pm0.04}$ |

[1]*DeepSeek-V2 is no more available when we add this baseline, therefore, we use DeepSeek-V2.5 instead.*

## E.5 Implementation of the FCI algorithm

We use a third-party open-sourced Python library to perform the FCI algorithm: https://causal-learn.readthedocs.io/en/latest/

We set $\alpha = 0.05$, and `independence_test_method="fisherz"` hroughout all experiments. Other parameters are kept as the default.

Table 6: Full Result of Causal Metrics each Round each LLM on Apple Gastronome Benchmark

| LLMs | COAT ROUND | PERCEPTION SCORE | CAPACITY SCORE | $I(y; x\|h_S)$ |
|---|---|---|---|---|
| GPT 4O | ITER 1 | 0.82±0.02 | 0.24±0.01 | 0.28±0.02 |
| | ITER 2 | 0.44±0.08 | 0.26±0.02 | 0.21±0.03 |
| | ITER 3 | 0.38±0.12 | 0.19±0.09 | 0.24±0.05 |
| GPT 4 | ITER 1 | 0.78±0.02 | 0.18±0.02 | 0.45±0.04 |
| | ITER 2 | 0.61±0.08 | 0.26±0.05 | 0.28±0.05 |
| | ITER 3 | 0.39±0.28 | 0.22±0.00 | 0.29±0.02 |
| GPT 3.5 | ITER 1 | 0.81±0.02 | 0.21±0.01 | 0.42±0.05 |
| | ITER 2 | 0.52±0.11 | 0.23±0.02 | 0.31±0.04 |
| | ITER 3 | 0.38±0.10 | 0.21±0.03 | 0.33±0.11 |
| MISTRAL-LARGE | ITER 1 | 0.78±0.02 | 0.20±0.01 | 0.41±0.04 |
| | ITER 2 | 0.61±0.08 | 0.20±0.03 | 0.29±0.02 |
| | ITER 3 | 0.33±0.33 | 0.32±0.00 | 0.25±0.05 |
| MISTRAL-MEDIUM | ITER 1 | 0.78±0.02 | 0.20±0.02 | 0.39±0.05 |
| | ITER 2 | 0.39±0.15 | 0.19±0.10 | 0.32±0.04 |
| | ITER 3 | 0.36±0.10 | 0.37±0.14 | 0.25±0.08 |
| LLAMA-3-70B | ITER 1 | 0.75±0.00 | 0.19±0.02 | 0.47±0.08 |
| | ITER 2 | 0.36±0.31 | 0.19±0.00 | 0.38±0.02 |
| | ITER 3 | 0.27±0.38 | 0.18±0.00 | 0.36±0.03 |
| LLAMA-2-70B | ITER 1 | 0.77±0.02 | 0.21±0.05 | 0.43±0.13 |
| | ITER 2 | 0.08±0.12 | 0.35±0.00 | 0.35±0.03 |
| | ITER 3 | 0.28±0.04 | 0.14±0.07 | 0.33±0.03 |
| QWEN-1.5-110B | ITER 1 | 0.77±0.02 | 0.17±0.01 | 0.45±0.09 |
| | ITER 2 | 0.56±0.16 | 0.21±0.12 | 0.32±0.08 |
| | ITER 3 | 0.28±0.21 | 0.18±0.02 | 0.33±0.11 |
| DEEPSEEK-V2 | ITER 1 | 0.89±0.08 | 0.17±0.02 | 0.42±0.05 |
| | ITER 2 | 0.62±0.10 | 0.20±0.02 | 0.34±0.02 |
| | ITER 3 | 0.67±0.24 | 0.40±0.12 | 0.29±0.06 |
| CLAUDE-3-OPUS | ITER 1 | 0.77±0.02 | 0.18±0.00 | 0.43±0.06 |
| | ITER 2 | 0.56±0.08 | 0.39±0.11 | 0.24±0.06 |
| | ITER 3 | 0.75±0.25 | 0.15±0.00 | 0.22±0.05 |

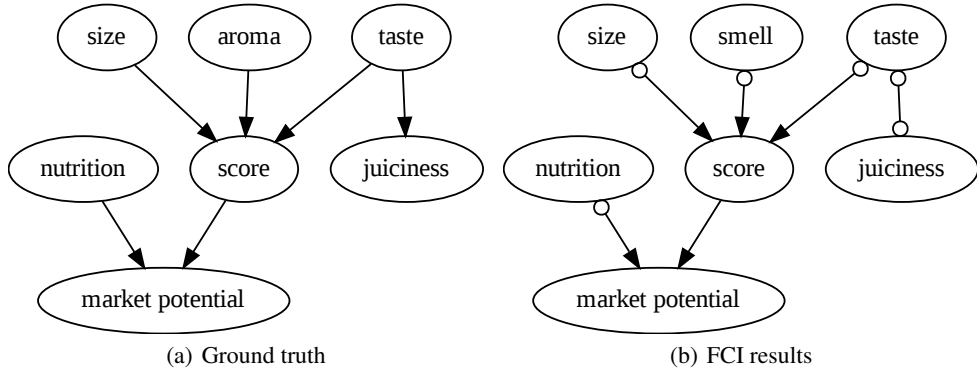

(a) Ground truth        (b) FCI results

Figure 14: Ground truth and faithful (via FCI algorithm) causal graphs in AppleGastronome.

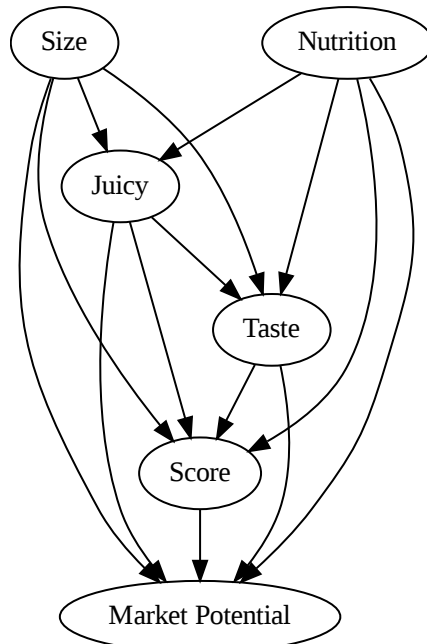

(a) GPT 4 reasoning

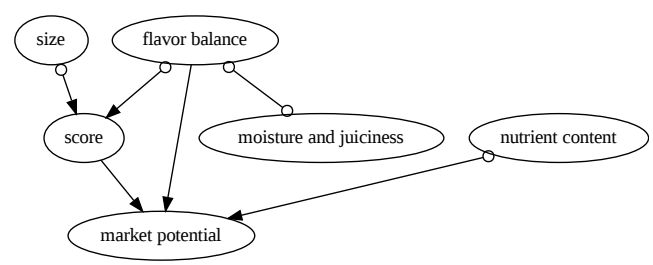

(b) GPT 4 COAT

Figure 15: Causal graphs with GPT 4 in AppleGastronome.

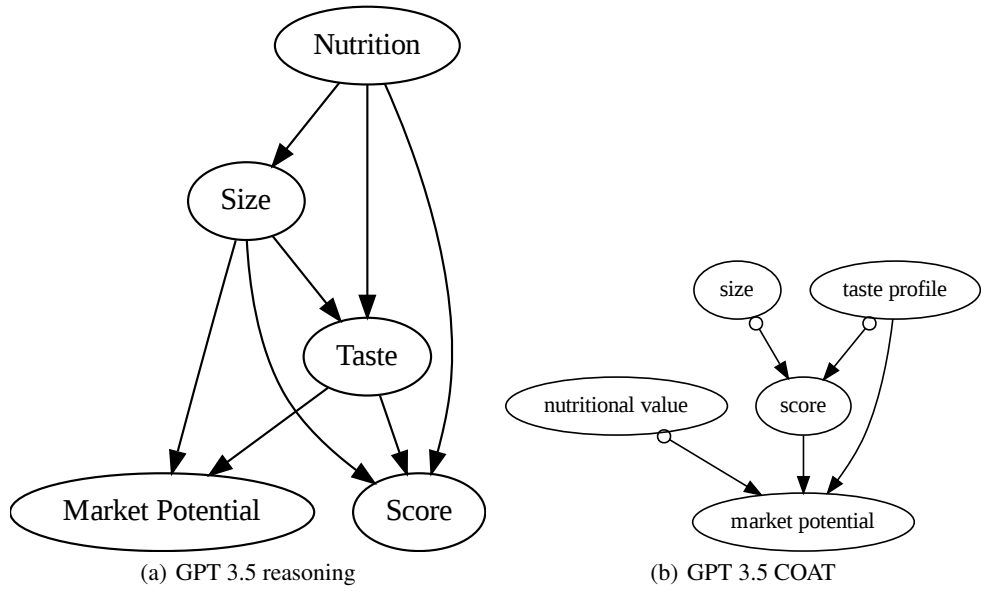

(a) GPT 3.5 reasoning                    (b) GPT 3.5 COAT

Figure 16: Causal graphs with GPT 3.5 in AppleGastronome.

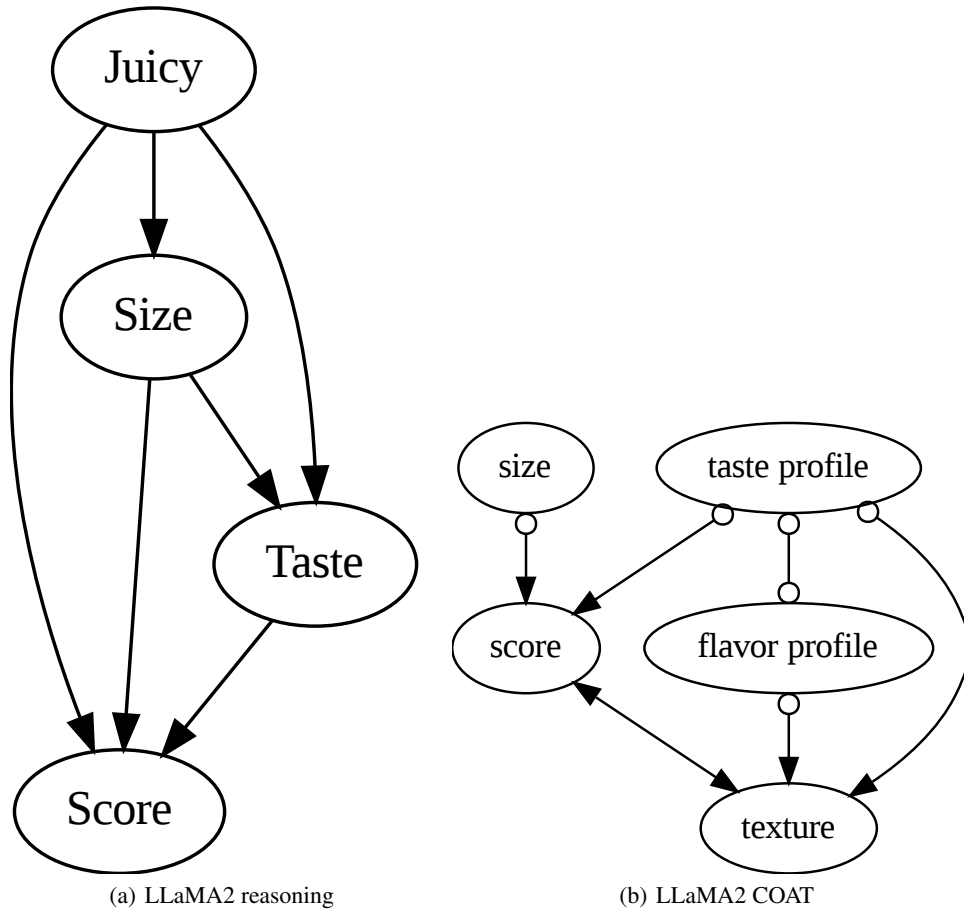

(a) LLaMA2 reasoning                    (b) LLaMA2 COAT

Figure 17: Causal graphs with Llama-2 in AppleGastronome.

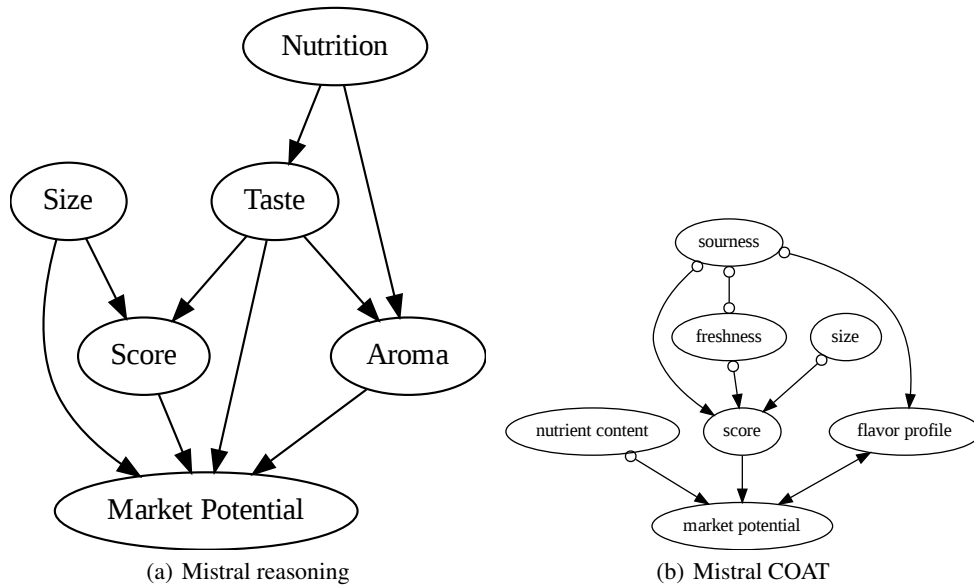

(a) Mistral reasoning        (b) Mistral COAT

Figure 18: Causal graphs with mistral Medium in AppleGastronome.

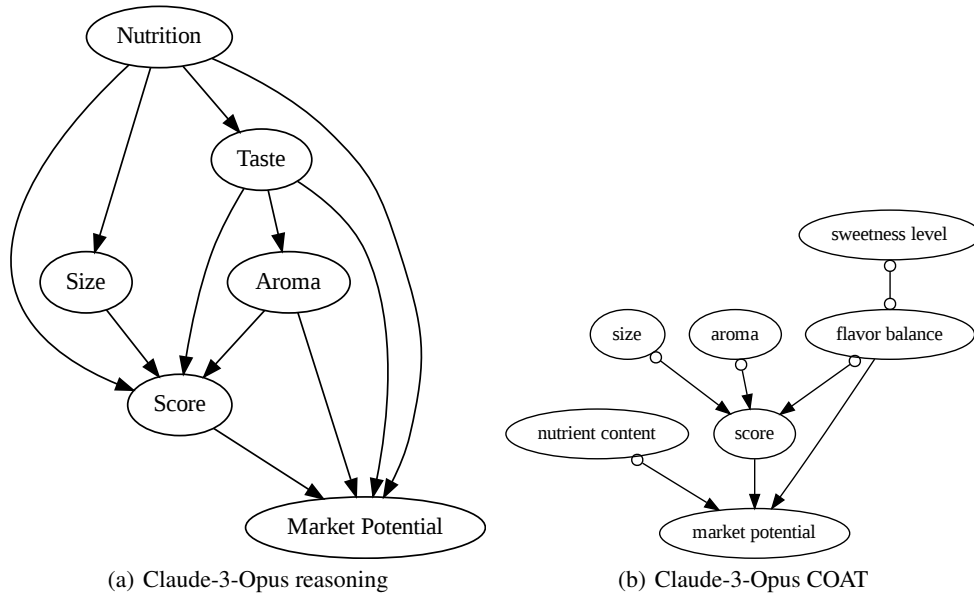

(a) Claude-3-Opus reasoning        (b) Claude-3-Opus COAT

Figure 19: Causal graphs with Claude-3-Opus in AppleGastronome.

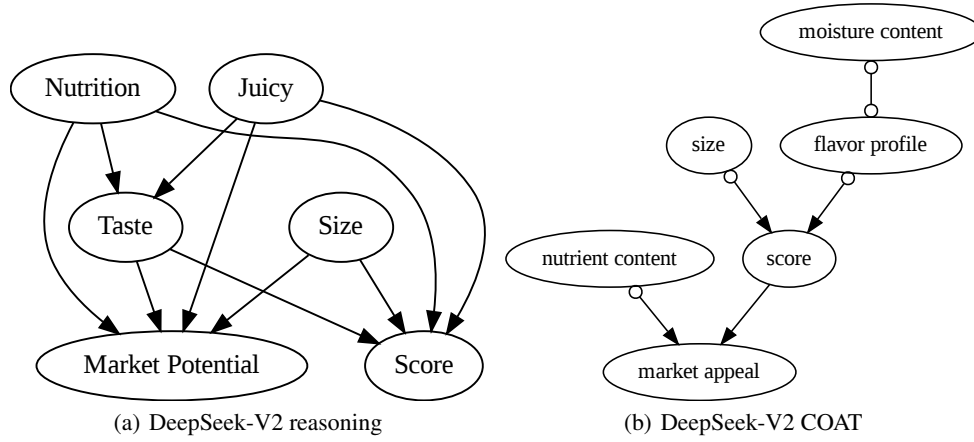

(a) DeepSeek-V2 reasoning

(b) DeepSeek-V2 COAT

Figure 20: Causal graphs with DeepSeek-V2 in AppleGastronome.

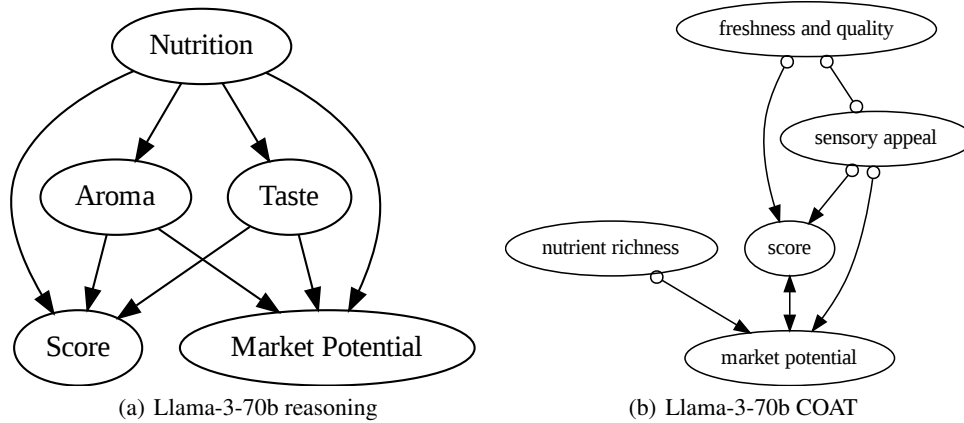

(a) Llama-3-70b reasoning

(b) Llama-3-70b COAT

Figure 21: Causal graphs with Llama-3-70b in AppleGastronome.

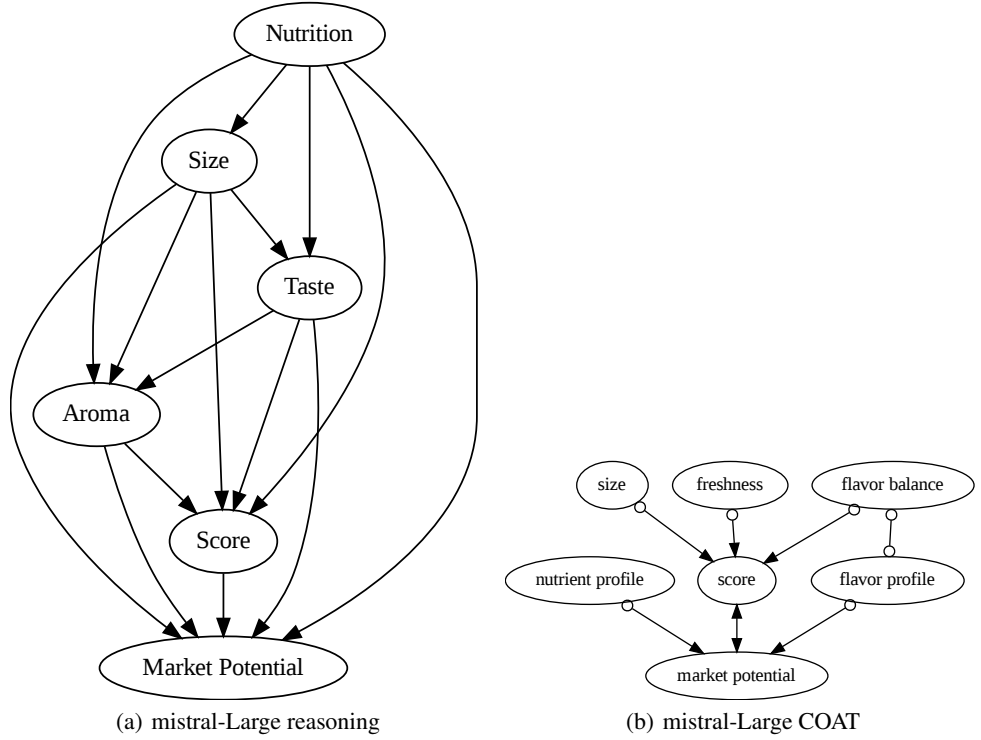

(a) mistral-Large reasoning

(b) mistral-Large COAT

Figure 22: Causal graphs with mistral-Large in AppleGastronome.

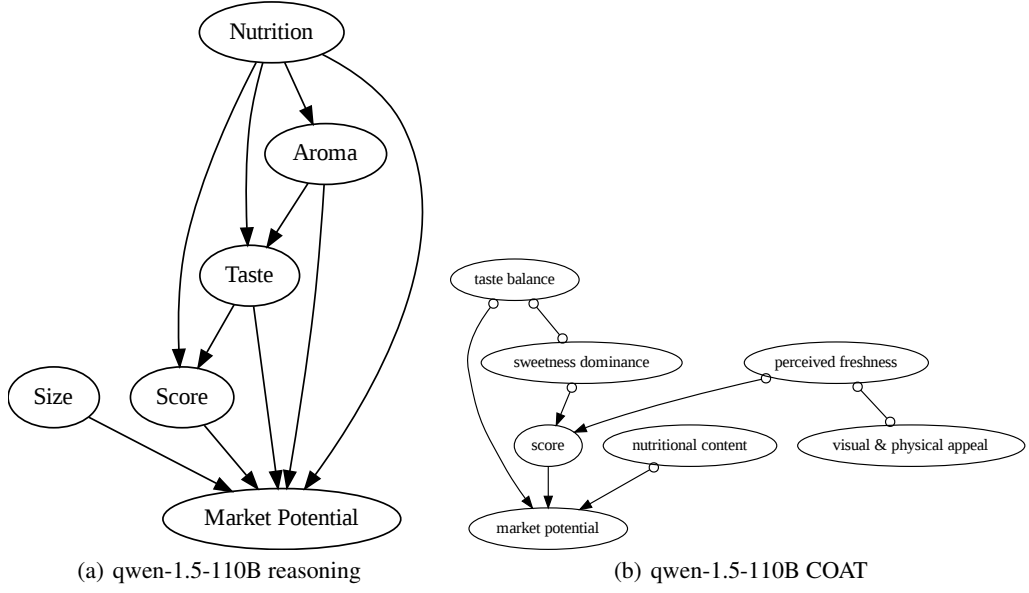

(a) qwen-1.5-110B reasoning

(b) qwen-1.5-110B COAT

Figure 23: Causal graphs with qwen-1.5-110B in AppleGastronome.

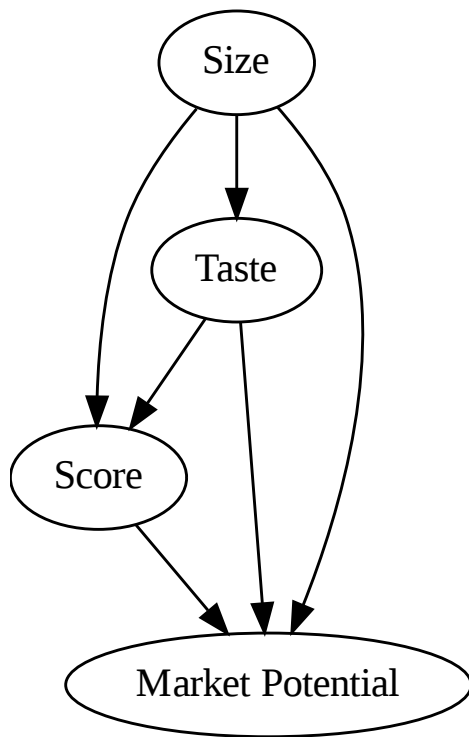

(a) GPT-4o reasoning

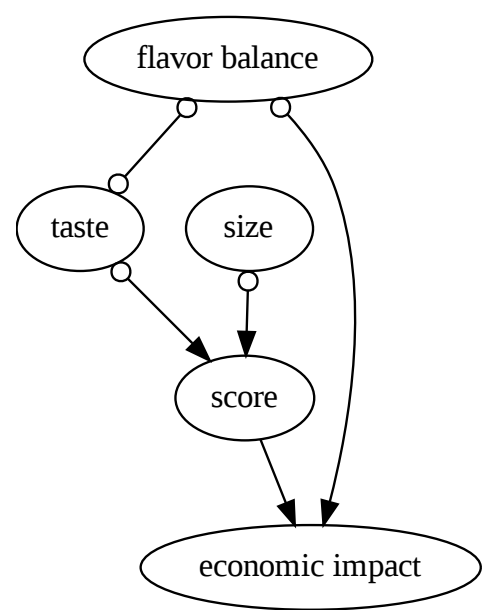

(b) GPT-4o COAT

Figure 24: Causal graphs with GPT-4o in AppleGastronome.

### E.6 More Details on Constructing Neuropathic

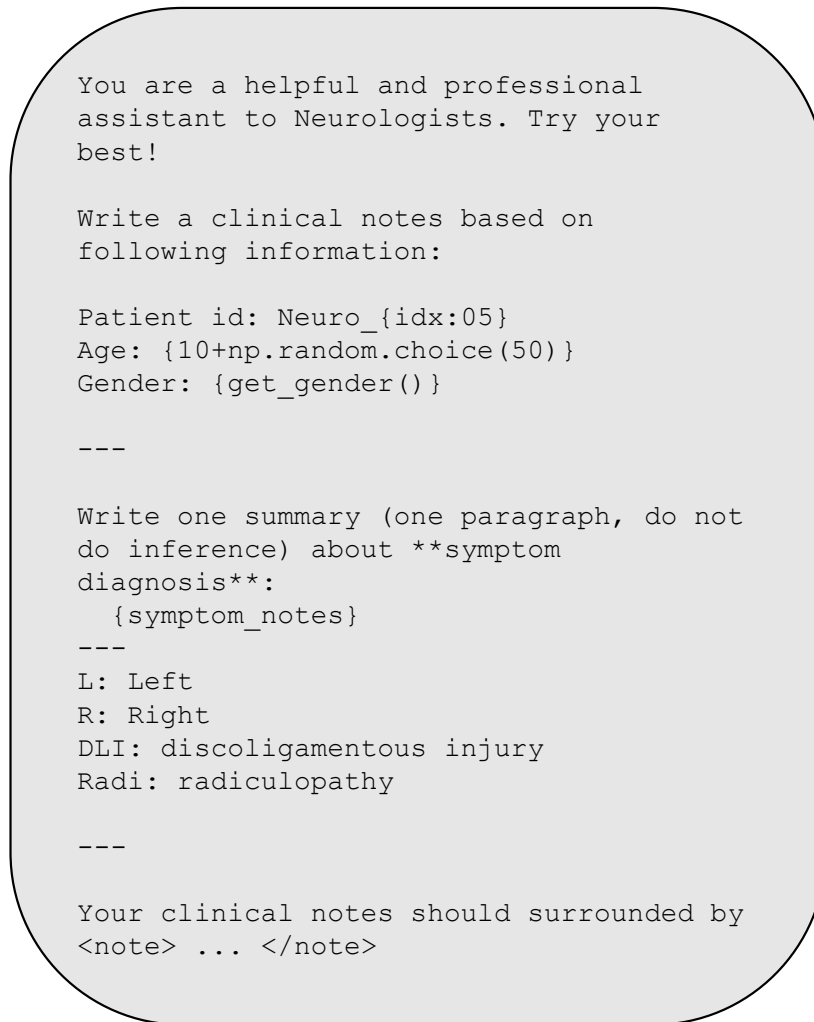

```
You are a helpful and professional
assistant to Neurologists. Try your
best!

Write a clinical notes based on
following information:

Patient id: Neuro_{idx:05}
Age: {10+np.random.choice(50)}
Gender: {get_gender()}

---

Write one summary (one paragraph, do not
do inference) about **symptom
diagnosis**:
  {symptom_notes}
---
L: Left
R: Right
DLI: discoligamentous injury
Radi: radiculopathy

---

Your clinical notes should surrounded by
<note> ... </note>
```

Figure 25: Illustration of prompts for generating Neuropathic.

In the Neuropathic benchmark, we convert the dataset into a clinical diagnosis task. In the original dataset, there are three levels of causal variables, including the symptom-level, radiculopathy-level and the pathophysiology-level. In experiments, we mainly consider the target variable of right shoulder impingement. When generating the clinical diagnosis notes as $x$ using GPT 4, we will avoid any mentioning of variables other than symptoms.

As we intend to leverage the Neuropathic benchmark to simulate the real-life diagnosis, after the factor proposal stage, we directly incorporate external experts that measure the values of the candidate factors. The prompts to generate the diagnosis records are given in Fig. 25.

Examples of Neuropathic are given in Fig. 26.

### E.7 More Details of Results on Neuropathic

The detailed causal graph results are given from Fig. 27 to Fig. 31.

### E.8 Discussion on the time complexity

Assume there are m samples with n possible factors.

```
## Group with 'R shoulder impingement'=0
----
Patient ID: Neuro_00000
Age: 57
Gender: Male

Symptom Diagnosis Summary:
The patient presents with a left shoulder impingement and associated intracapsular problems. The
symptoms include limited and painful range of motion with a positive outcome when performing a
"thumbs up" test on the left side. Right upper limb assessment also reveals issues; the patient
wears an armband suggesting ongoing management for right lower arm disorders. The patient
reports lumbago and pain localized to the left thigh. Physical therapy assessment (L pta) is
noted, indicating involvement in a therapeutic intervention. Joint assessments reveal a left hip
joint condition and right hip arthritis, accompanied by right hamstring discomfort.
```

Figure 26: Illustration of examples in Neuropathic.

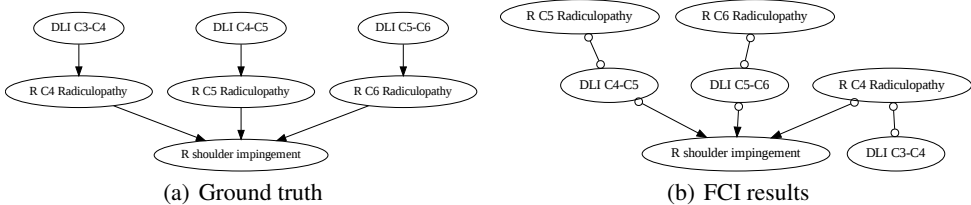

(a) Ground truth                    (b) FCI results

Figure 27: Ground truth and faithful (via FCI algorithm) causal graphs in Neuropathic.

For Factor Proposal:

META only needs to interact once with LLM, so it is O(1). DATA runs one single COAT round, so it is O(1). COAT interacts with LLMs multiple rounds. In each round, at least one new factor should be proposed; otherwise, the loop will stop. So it is O(n) For Factor Annotation:

META: Not applicable. DATA: At most n factors would be proposed in a single round. And each of them needs m times annotations by LLMs for all samples. So it is O(nm). COAT: At most n factors would be proposed during all rounds. And each of them needs m times annotations by LLMs for all samples. So it is O(nm). For Causal Discovery:

Pair-wise reasoning by LLMs: $O(n^2)$ COAT: LLM is not involved. The computational cost of it depends on the numeric methods. The FCI algorithm used by COAT has a time complexity that goes exponentially with n. Learning the causal graph over a large number of nodes effectively is still an open problem in causal discovery literature.

## E.9 Resources

We utilized a system comprising two Intel Xeon E5-2630v4 processors with 2.2GHz, two NVIDIA Tesla P40 GPUs, and 256 GB of memory. For conversations with large language models (LLMs), we leveraged the poe.com platform, while annotations were facilitated using the OpenAI API and the Mistral API.

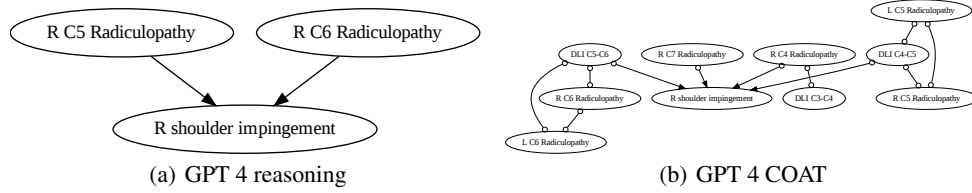

(a) GPT 4 reasoning        (b) GPT 4 COAT

Figure 28: Causal graphs with GPT 4 in Neuropathic.

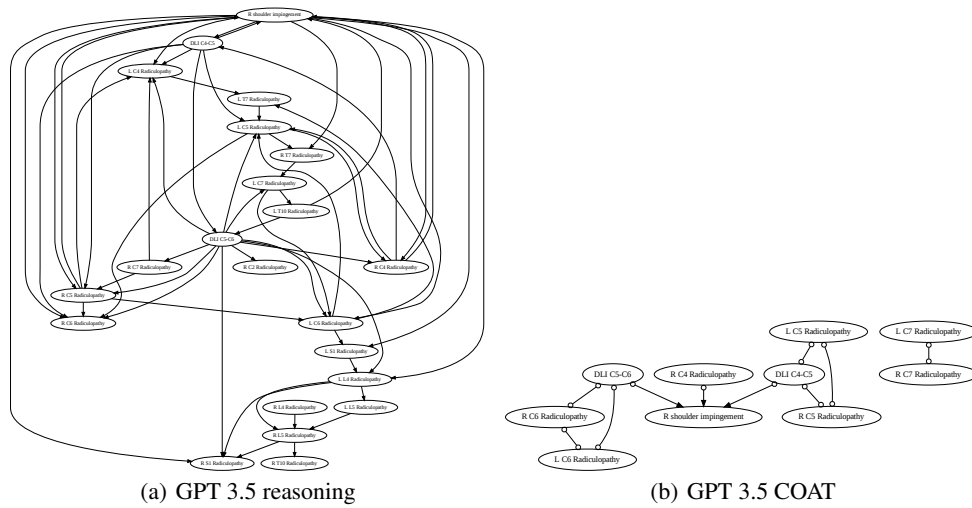

(a) GPT 3.5 reasoning        (b) GPT 3.5 COAT

Figure 29: Causal graphs with GPT 3.5 in Neuropathic.

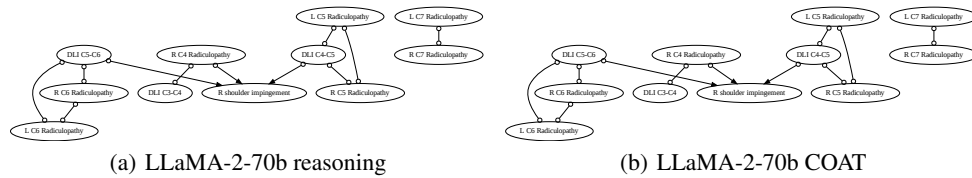

(a) LLaMA-2-70b reasoning        (b) LLaMA-2-70b COAT

Figure 30: Causal graphs with LLaMA-2-70b in Neuropathic.

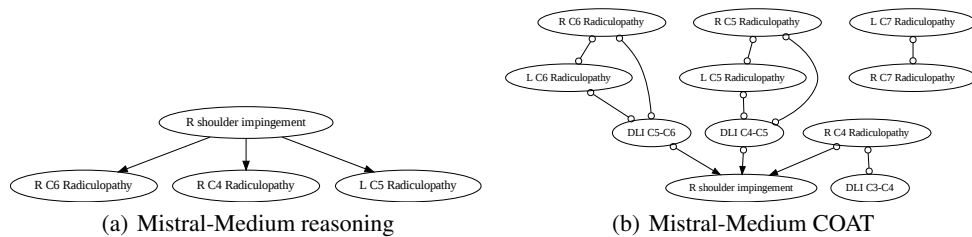

(a) Mistral-Medium reasoning        (b) Mistral-Medium COAT

Figure 31: Causal graphs with Mistral-Medium in Neuropathic.

# F COAT with Different Causal Discovery Algorithm

Table 7: Causal discovery results in Neuropathic. PA, AN, and OT refer to the parents, ancestors, and others, respectively. Accuracy and F1 measure the recovery of the causal ancestors.

| LLM | METHOD | FACTOR PROPOSAL | | | | |
| --- | --- | --- | --- | --- | --- | --- |
| | | PA | AN | OT | ACC | F1 |
| GPT 4 | META | 3 | 5 | 6 | 0.91 | 0.59 |
| | DATA | 2 | 2 | 0 | 0.95 | 0.50 |
| | COAT | 3 | 6 | 3 | 0.96 | 0.80 |
| | DATA (LINGAM) | 3 | 3 | 0 | 0.96 | 0.67 |
| | COAT (LINGAM) | 3 | 6 | 4 | 0.95 | 0.75 |
| GPT 3.5 | META | 3 | 5 | 6 | 0.91 | 0.59 |
| | DATA | 3 | 5 | 4 | 0.94 | 0.67 |
| | COAT | 3 | 5 | 2 | 0.96 | 0.77 |
| | DATA (LINGAM) | 2 | 3 | 4 | 0.91 | 0.46 |
| | COAT (LINGAM) | 3 | 5 | 4 | 0.94 | 0.67 |
| LLAMA2-70B | META | 2 | 4 | 5 | 0.91 | 0.53 |
| | DATA | 3 | 3 | 1 | 0.95 | 0.60 |
| | COAT | 3 | 6 | 2 | 0.97 | 0.86 |
| | DATA (LINGAM) | 3 | 4 | 4 | 0.92 | 0.57 |
| | COAT (LINGAM) | 3 | 6 | 4 | 0.95 | 0.75 |
| MISTRAL-MED | META | 3 | 6 | 3 | 0.96 | 0.80 |
| | DATA | 3 | 3 | 2 | 0.94 | 0.66 |
| | COAT | 3 | 6 | 2 | 0.97 | 0.86 |
| | DATA (LINGAM) | 3 | 3 | 3 | 0.92 | 0.50 |
| | COAT (LINGAM) | 3 | 6 | 3 | 0.96 | 0.80 |

# G Ablation Study

**Group Size in Prompt.** In the COAT prompt, several samples are given and grouped by the values of the target variable. The samples in each group are randomly selected to a fixed number (like 3 samples per group). Empirically, we keep it to be 3 throughout all experiments (sometimes smaller than 3 if samples are not enough). In practice, it is mainly constrained by the LLM's context length.

**The Number of Clusters.** When constructing feedback, we first use clustering to separate the dataset and then find the cluster where the target variable is not explained well by current factors (This is a heuristic for the problem in line 191). Empirically, we set the number of clusters to be one plus the number of current factors.

We conduct ablation studies of COAT with GPT-4 using different hyperparameters:

Table 8: Results about Ablation Study on Hyperparameters.

| Method | Cluster Size | Group Size | MB | NMB | OT | Recall | Precision | F1 |
| --- | --- | --- | --- | --- | --- | --- | --- | --- |
| META | - | - | $2.67_{\pm0.94}$ | $0.67_{\pm0.47}$ | $2.33_{\pm0.47}$ | $0.53_{\pm0.19}$ | $0.46_{\pm0.08}$ | $0.49_{\pm0.13}$ |
| DATA | - | 3 | $3.00_{\pm0.00}$ | $0.33_{\pm0.47}$ | $0.00_{\pm0.00}$ | $0.60_{\pm0.00}$ | $0.92_{\pm0.12}$ | $0.72_{\pm0.04}$ |
| COAT | len(factor)+1 | 3 | $4.00_{\pm0.82}$ | $0.33_{\pm0.47}$ | $0.00_{\pm0.00}$ | $0.80_{\pm0.16}$ | $0.93_{\pm0.09}$ | $0.85_{\pm0.11}$ |
| COAT | len(factor)+1 | 1 | $4.67_{\pm0.58}$ | $0.00_{\pm0.00}$ | $0.00_{\pm0.00}$ | $0.93_{\pm0.12}$ | $1.00_{\pm0.00}$ | $0.96_{\pm0.06}$ |
| COAT | 2 | 3 | $3.67_{\pm1.53}$ | $0.00_{\pm0.00}$ | $0.00_{\pm0.00}$ | $0.73_{\pm0.31}$ | $1.00_{\pm0.00}$ | $0.82_{\pm0.22}$ |

As shown in Table 8, one can observe that COAT is not sensitive to these hyperparameters and performs robustly well than the baselines under different hyperparameter setups.

**Prompt Template.** We conduct an ablation study with a different prompt template following [101]:

- We put the task description (also the format instructions) in the beginning `[System]` part, and we put samples in the last `[Data]` part of the prompt.
- The markdown grammar is replaced by blankets to represent headings, like `[System]`, `[Data]`, and `[Groups with Y=1]` ...
- 3 COAT iterations are performed, which is aligned with the original experimental setup.

TABLE 9: COAT WITH CHANGED PROMPT TEMPLATE

| MODEL | MB | NMB | OT | RECALL | PRECISION | F1 |
|-------|----|-----|----|----|----|----|
| GPT-4 | 4 | 0 | 0 | 0.80 | 1.00 | 0.89 |
| GPT-3.5-TURBO | 4 | 0 | 0 | 0.80 | 1.00 | 0.89 |
| MISTRAL-MEDIUM | 3 | 0 | 0 | 0.60 | 1.00 | 0.75 |

In Table 9, we observe that COAT is robust to the choice of templates, rejects unexpected factors (zero NMB and OT), and keeps a high precision.

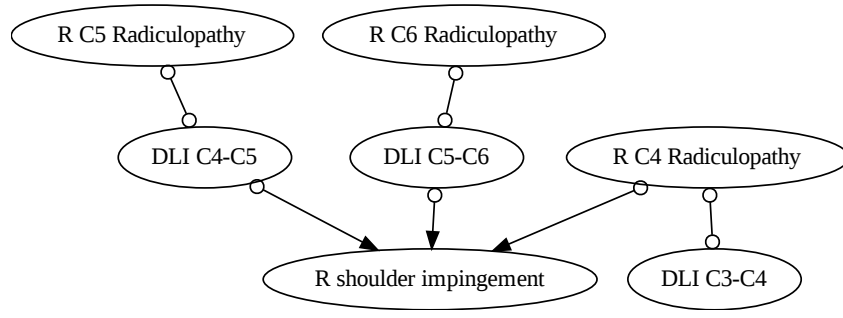

(a) Faithful ground truth

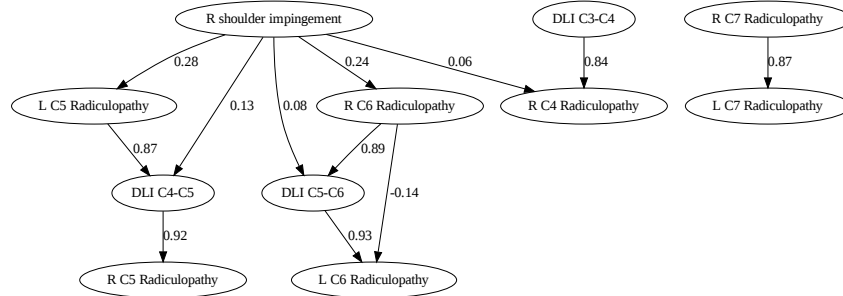

(b) GPT-4 COAT with LiNGAM

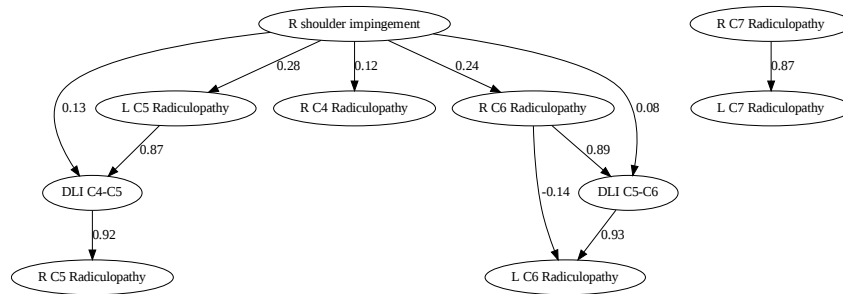

(c) GPT-3.5 COAT with LiNGAM

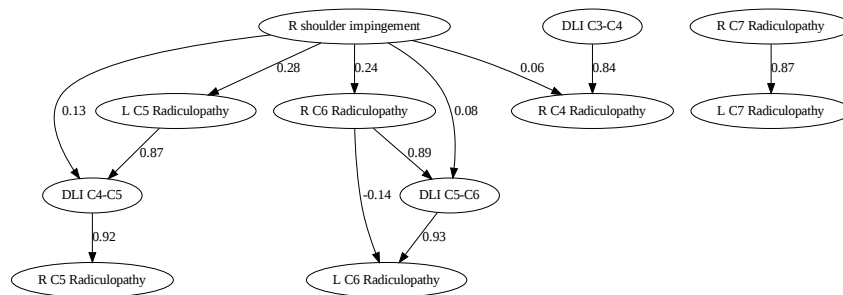

(d) llama-2-70b COAT with LiNGAM

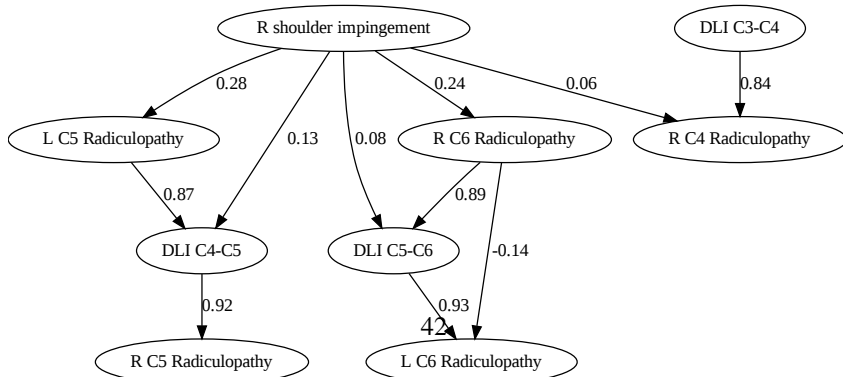

(e) Mistral-med COAT with LiNGAM

# H   Summary of Benchmark Data

| Benchmark Name | Type | Source | Sample | Sample Size | Ground Truth |
|---|---|---|---|---|---|
| Apple Gastronome | Synthetic Data | This paper. It will be open-sourced under CC-BY 4.0. | Textual | 200 textual reviews. | Yes |
| Neuropathic | Semi Real-world Data | Tabular samples from Tu et al. [26] on their GitHub repo under CC-BY 4.0. Textual samples are generated by this paper. | Textual | 100 synthetic textual data for prompt construction; 1000 tabular data for CI tests. | Yes |
| Brain Tumor | Real-world Data | An open Kaggle dataset (kaggle/brain-tumor-classification-mri) with an open-sourced project. [38]. MIT License. | Image | 60 MRI samples for COAT and CI tests. The sample size is small due to the expensive API cost. One can easily construct a much larger dataset by following our instructions. | No |
| Stock News | Real-world Data | An open Kaggle dataset (kaggle/stock-price-and-news-realted-to-it) [39]. MIT License. | Textual with time index | 200 samples for COAT experiments. 400 samples for empirical validation. | No |
| ENSO | Real-world Data | The NOAA 20th Century Reanalysis V3 dataset [40] from their website at https://psl.noaa.gov (with CC0 1.0 License). We developed a set of simple tools for LLMs to utilize the official database. | NetCDF | Monthly observation for 1836/01 to 2015/12; 1.0 degree latitude x 1.0 degree longitude global grid (360x181); 90+ climate variables. | No |

Table 10: Summary of Benchmark Data

# I   Case Study on Brain Tumor

In this section, we utilize the COAT to explore the image dataset.

## I.1   The Brain Tumor Dataset

**Dataset**   Magnetic Resonance Imaging (MRI) is an important technique for detecting tumors in the human brain. The images are from an open Kaggle dataset (kaggle/brain-tumor-classification-mri) with an open-sourced project. [38]. Each sample is a scanning MRI of a human brain. In this case study, the interesting variable is the tumor type. We consider three types of MRI images: glioma, meningioma, and no tumor. We include 20 images for each category and the total sample size is 60.

**Data Processing**   We use gpt-4-vision-preview to handle image samples. As the current gpt4 cannot process multiple images simultaneously, we concatenate samples from different categories into one picture, as shown in Fig. 33, and give instructions in the prompt to explain the format, as shown in Fig. 34. For each proposed factor, the LLMs would go through all 60 sample images individually to evaluate the factor value.

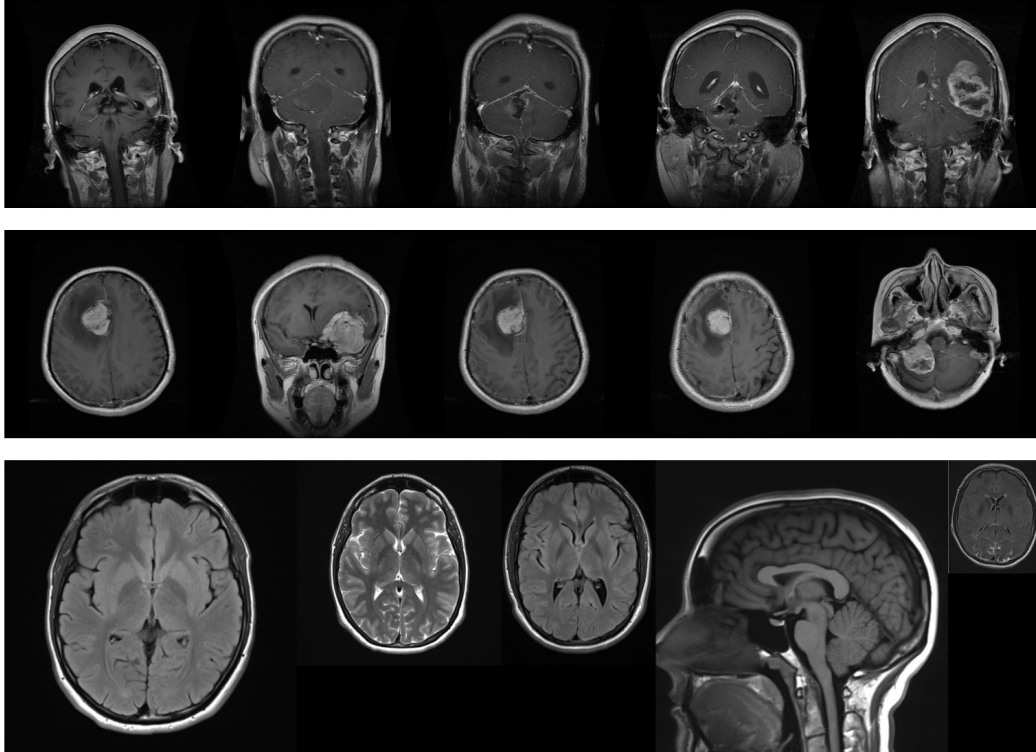

Figure 33: The initial input to COAT in the Brain Tumor case study. Each row contains 5 samples randomly selected from one category (top-down: glioma, meningioma, and no tumor).

**Evaluation**   The ground truth is not directly available to this dataset. Therefore, each factor in the final result will be evaluated by searching the related literature and will be explained by human interpretation.

## I.2   Result and Discussion

**Result interpretation**   As shown in Fig. 35, there are two visual factors that appeared in the final causal graph: *contrast enhancement* and *mass effect*. The first factor (contrast enhancement) is about whether the bright part is uniform or heterogeneous in the sample image. Another factor (mass effect) is about whether the tumor tissue has caused significant displacement of normal brain structures. One may refer to Fig. 36 to see detailed descriptions of these two factors.

**Justification for the final factors**   We verify the proposed factors by the medical literature [102, 103, 104, 105], as shown Fig. 37. Note that these papers are searched according to keywords in factor descriptions and, therefore, can exclude some relevant studies. Therefore, the conclusions of this preliminary exploration should be considered a reference point for further in-depth validation by domain experts.

The factor description for *contrast enhancement* directly matches with the related paper (the first two papers displayed in Fig. 37). In addition, this can be visually checked in Fig. 33. Therefore, we believe it is a good factor.

The second factor, *mass effect*, may not directly match the papers. It is pointed out in the last two papers displayed in Fig. 37 that the two tumors have different axial locations and thus influence the magnitude of the displacement of tissues. Both the keywords "axial location" and "displacement" have occurred in its factor description.

```
{Uploaded image}
You are an excellently helpful AI assistant for analyzing, abstracting, and processing data. Now try
your best.

# Data

## Group 1: glioma tumor
See the 5 samples in the 1st row in the picture.

## Group 2: meningioma tumor
See the 5 samples in the 2nd row in the picture.

## Group 3: no tumor
See the 5 samples in the 3rd row in the picture.

# Tasks: Factor Abstraction.

**What are the high-level factors behind the sample that contribute to the allocation of groups?**
……
```

Figure 34: Illustration of the prompt for the Brain Tumor case study. It contains instructions about understanding the combined input picture.

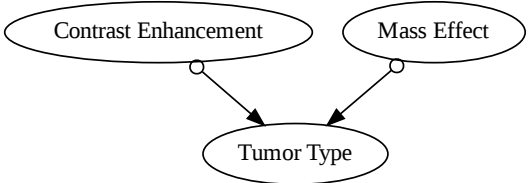

Figure 35: Final causal graph by COAT in the Brain Tumor case study.

# J    Case Study on Stock News

In this section, we utilize COAT to explore time series data with text.

## J.1    The Stock News Dataset

**Dataset**    The dataset consists of the stock value of Microsoft (MSFT) from 2006 to 2009 and its news summary (only include news in *the New York Times*). This is a subset of an open Kaggle dataset (kaggle/stock-price-and-news-realted-to-it). Each sample is one trading day with the company's close stock price and news. The target value is the future return rate, and we are curious about factors in the related news. We fed data during the first 200 trading days to COAT, and we used the following 400 trading days for evaluation.

**Data processing**    The future return rate is calculated by close prices:

$$r_t = \frac{\text{close}_{t+4}}{\text{close}_t} - 1. \tag{24}$$

The target value is a binary variable according to the return rate

$$Y_t = \mathbf{1}_{r_t > 0}, \tag{25}$$

where $\mathbf{1}_A$ is the indicator function. News fed to COAT are grouped according to the value of $Y_t$. To keep the whole prompt within the context limit, only 3 samples would be randomly included in each group.

**Factor processing**    Given one proposed factor, denote its annotated value at the $t$-th trading day as $a_t \in \{-1, 0, 1\}$. Each factor is rolling averaged over the past $M$ days:

$$S_t = \frac{1}{M} \sum_{t-M \geq i \geq t} a_i. \tag{26}$$

```
Contrast Enhancement

• 1: Strong, uniform enhancement typical of a meningioma.

• 0: No abnormal enhancement or no mass present.

• -1: Heterogeneous enhancement with or without areas of necrosis typical of a glioma.

Mass Effect

• 1: Mass causing significant displacement of normal brain structures (can occur with both meningioma and glioma but is more pronounced in glioma
  due to intra-axial location).

• 0: No mass is present, or the specific criteria for 1 or -1 are not met.

• -1: Mass causing minimal or no displacement of brain structures (more common in meningioma due to extra-axial location).
```

Figure 36: Detailed descriptions of these two final factors in the Brain Tumor case study. These descriptions were also directly fed to LLMs for factor annotation on images.

"They (meningiomas) almost always demonstrate uniform strong enhancement on post-contrast imaging ... "
*Lyndon, Daniel, et al. "Dural masses: meningiomas and their mimics." Insights into imaging 10.1 (2019): 11.*

"High-grade (gliomas ) tumors were heterogeneous contrast enhancement with other medical imaging aspects of MRI such as bleeding, necrosis, and edema"
*Haydar, Nisreen, et al. "Role of Magnetic Resonance Imaging (MRI) in grading gliomas comparable with pathology: A cross-sectional study from Syria." Annals of Medicine and Surgery 82 (2022): 104679.*

"Glioblastoma multiforme (GBM) is the most common primary brain tumor and is associated with a poor prognosis. One of its defining characteristics is the significant deformation of surrounding tissue, the so-called "mass effect".
*Tunç, Birkan, et al. "Modeling of glioma growth with mass effect by longitudinal magnetic resonance imaging." IEEE Transactions on Biomedical Engineering 68.12 (2021): 3713-3724.*

"Meningiomas typically appear as lobular, extra axial masses with well-circumscribed margins. They typically have a broad-based dural attachment and, if sufficiently large, inward displacement of the cortical grey matter. "
*Watts, J., et al. "Magnetic resonance imaging of meningiomas: a pictorial review." Insights into imaging 5 (2014): 113-122.*

Figure 37: Medical literature about the brain tumor. Searched according to keywords in factor descriptions.

Each variable, including return rate, is normalized by the rolling standard deviation over the past $M$ days:

$$F_t = \frac{1}{\text{Sd}\left(\{S_i\}_{t-M \geq i \geq t}\right) + 1} S_t. \tag{27}$$

An example of a processed factor can be seen in Fig. 38.

## J.2 Result and Discussion

**Result Causal Graph** As shown in Fig. 39, four factors are identified in the final causal graph to form a Markov blanket of return rate: *Product Focus*, *Legal and Regulatory Issues*, *Market Strategy*, and *Innovation and Technology Focus*. One can refer to Fig. 40 for more detailed descriptions of these factors. One factor (*Innovation and technology focus*) is identified as a possible cause of the return rate; this matches the nature of the company's type and reflects people's expectation of the company to keep creating innovative computer software. It is also interesting to see that COAT captures the structure between factors and *market strategy*, where *product focus* and *legal and regulatory issues* are identified as potential causes. It also implies the existence of a latent confounder between *market strategy* and *return rate* that may not be significantly reflected in news text.

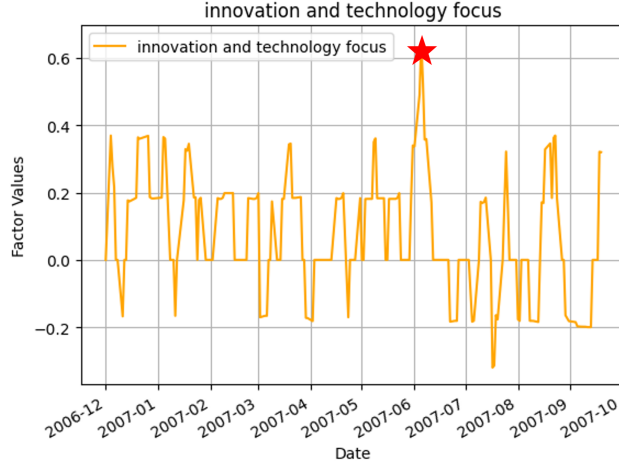

Figure 38: Example of one processed factors *Innovation and Technology Focus* during the first 200 trading days. The red star marks the highest value. In June 2007, there was a discussion about future competition in desktop operating systems and the trend towards web-based services [106].

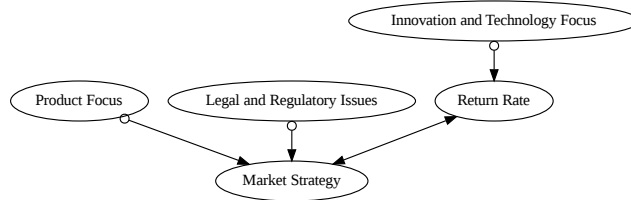

Figure 39: The final causal graph by COAT in the Stock News case study

**Evaluation by Trading Strategies**  For each factor $\{F_t\}$ processed in Eq. 27, we establish a trading strategy on it to see the performance in the out-of-sample trading days. At each trading day $t$, we fit the following model

$$r_t = \alpha + \beta F_t, \tag{28}$$

using samples $\{r_i, F_i\}_{t-200 \le i < t}$. Note that $r_t$ is the future return rate defined in Eq. 24 and is not available on this day. After the estimation, we make decision based on $\hat{r}_t := \hat{\alpha} + \hat{\beta} F_t$. If $\hat{r}_t > 0$, we *go long* with 1 unit capital, i.e., purchase that amount of stock, the gain would be gain would be $r \times 1$ units capital. If $\hat{r}_t < 0$, we *go short* with 1 unit capital, i.e., borrow and sell that amount of stocks immediately and buy them back next time, the gain would be $-r \times 1$ units capital.

To align with the definition in Eq. 24, we make trading decisions every $4$ trading day. We introduce an additional *Buy and Hold* baseline to always go long one unit capital. The trading evaluation is after the 200-th day and thus has no overlapping with data fed to COAT.

The cumulative return plot is shown in Fig. 41, and the metrics commonly used in economic literature [107, 108, 109] are shown in Table 11. One important metric to see a trading strategy's effectiveness is the sharp ratio: a measure of risk-adjusted return, showing the excess return (over the risk-free rate, we set it to be $2\%$) per unit of standard deviation. A higher Sharpe ratio indicates better performance per unit of risk. We see that the *Innovation and Technology Focus* factor yields the highest sharp ratio and outperforms other non-causal factors.

```
Product Focus
•   - 1: Text primarily discusses the launch, performance, or features of a company's product.
•   - 0: Otherwise; or not mentioned.
•   - -1: Text does not focus on products but rather on other aspects such as company strategy, business practices, or financial results.

Legal and Regulatory Issues
•   - 1: Text is centered around legal proceedings, regulatory actions, or oversight related to a company's business practices.
•   - 0: Otherwise; or not mentioned.
•   - -1: Text does not discuss legal or regulatory issues but focuses on other topics like product information or market performance.

Market Strategy
•   - 1: Text discusses company strategies related to market expansion or competitive actions.
•   - 0: Otherwise; or not mentioned.
•   - -1: Text focuses on internal company issues, product details, or legal/regulatory matters without a direct link to market strategy.

Innovation and Technology Focus
•   - 1: Articles highlighting new technologies, innovations, or significant advancements in existing technology.
•   - 0: Otherwise; or not mentioned.
•   - -1: Articles discussing established technologies, standard updates, or non-innovative topics.
```

Figure 40: The detailed factor descriptions proposed by COAT in the Stock News case study

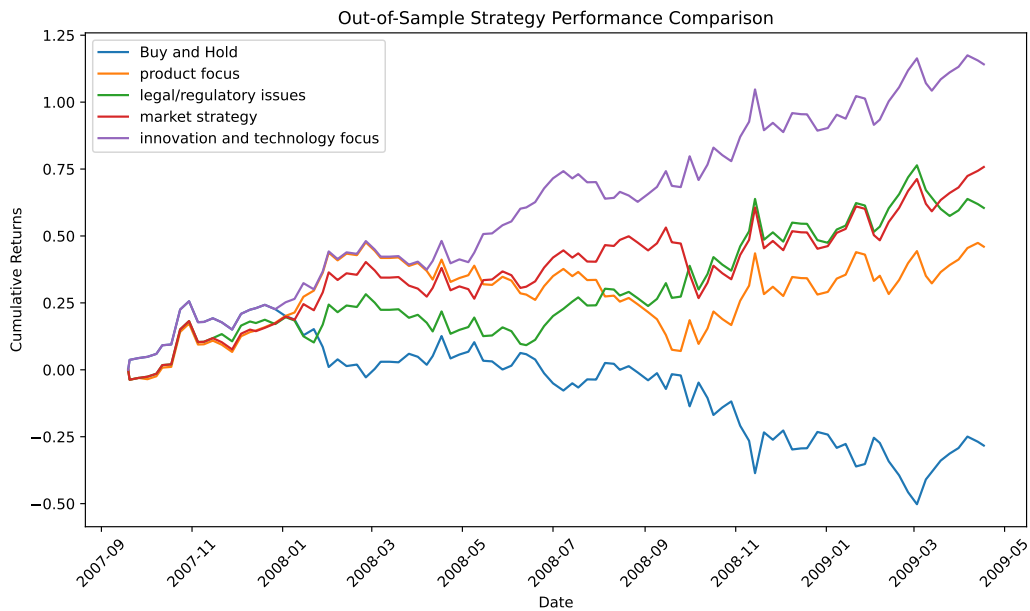

Figure 41: The out-of-sample cumulative return of trading strategies based on different factors in the Stock News case study

Table 11: Performance about trading strategy according to each factors

|  | BUY AND HOLD | PRODUCT FOCUS | LEGAL/REGULATORY ISSUES | MARKET STRATEGY | INNOVATION AND TECHNOLOGY FOCUS |
|---|---|---|---|---|---|
| EXPECTED RETURN | -3.50 | 0.29 | 0.38 | 0.48 | 0.72 |
| SHARP RATIO | -2.38 | 0.70 | 0.94 | 1.20 | **1.86** |
| T-STAT | -3.00 | 0.89 | 1.19 | 1.51 | 2.35 |
| INFORMATION RATIO | - | 2.45 | 2.50 | 2.56 | 2.73 |
| $\alpha$ | - | 3.79 | 3.88 | 3.98 | 4.22 |
| $\alpha$ T-STAT | - | 3.09 | 3.15 | 3.23 | 3.44 |
| MAX LOSS | -0.50 | -0.15 | -0.15 | -0.15 | -0.15 |
| SKEW | -0.43 | -0.14 | -0.15 | -0.41 | -0.38 |

# K   Case Study on El Niño–Southern Oscillation (ENSO)

In this section, we use COAT to explore a scientific problem using open datasets in climate science. The case study shows its potential to handle more complex data formats when combined with tools.

**The ENSO Background**    El Niño-Southern Oscillation (ENSO) is an important phenomenon in the Pacific Ocean and is characterized by irregular fluctuations in sea surface temperatures[110]. ENSO consists of the warm phase (known as *El Niño*) and the cold phase (known as *La Niña*). It profoundly influences global weather patterns, including precipitation, storm development, and temperature anomalies [111]. The prediction of ENSO events involves the complex interplay between oceanic and atmospheric systems and, therefore, is still an open problem [112, 113].

**NOAA Dataset**    The NOAA 20th Century Reanalysis V3 dataset contains high-dimensional information about Earth's atmosphere with fine-grained time and space coverage from the 19th century to the early 21st century [40]. This dataset amalgamates many observational data and uses the ensemble filter data assimilation method to reconstruct the historical state of the global atmosphere. The dataset includes a fine-grained spatial coverage with $360 \times 181$ grids. We only use monthly data in this case study.

## K.1   Setting and Data processing

**The Target Variable Construction**    The focus of this study is the future change in monthly SST in the Nino3 region, which could be an important indicator of ENSO events. The target variable is manually crafted from the dataset using API and is visualized in Fig. 42, where the oscillation pattern can be seen.

```
# nino3 sst
y = Observation(
        measurement='Skin Temperature',
        level='Surface',
        region={'latitude_min': -5, 'latitude_max': 5, 'longitude_min': 210, 'longitude_max': 270}
    )

# next month shift
y = y.shift(1) - y
```

Listing 1: Code to define the target variable in ENSO case study

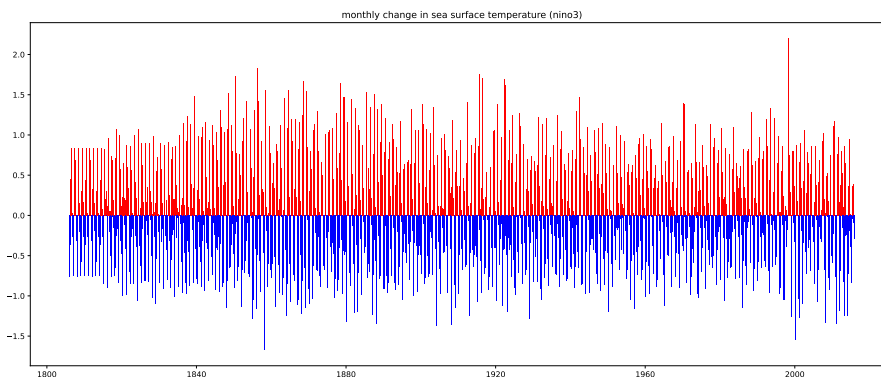

Figure 42: The visualization of the target variable defined in the ENSO case study. The oscillation pattern can be seen.

**Prepare Tools for LLMs**    The dataset is in the NetCDF format (network Common Data Format), so it is not convenient to be directly fed to LLMs. We prepare a function Observation(measurement

, `level`, `region`) to help define factors. For example, in Lst. 1, we define the target variable with the help of this function. The *measurement* specifies the climate variables like precipitation rate or temperature; the *level* specifies the vertical location, like surface or specific pressure level; the region is a rectangle about the area concerned. This function would output a single time series about the measurement on the specified level averaged within the specified region. Note that the target variable is the only factor defined by humans, all other factors are defined by code from LLMs.

```python
def get_air_temperature_at_cloud_layer_top_nino3():
    '''
    Air Temperature at Cloud Layer Top (Nino3 Region)
    '''
    # Declare observation
    observation_name = Observation(
        measurement="Air Temperature",
        level="Isentropic Levels",
        region={'latitude_min': -5, 'latitude_max': 5, 'longitude_min': 210, 'longitude_max': 270},
        detailed_level=300
    )

    # Return the observation values directly
    return observation_name

factor_dict['Air Temperature at Cloud Layer Top (Nino3 Region)'] = get_air_temperature_at_cloud_layer_top_nino3()
```

Listing 2: LLM can be prompted to propose factors using complex tools

**Feedback Construction**   The dataset is in a special format; we can only draw a tabular subset of data from it in this setting. Therefore, it is difficult to make groups on samples like what we did in the previous cases. Instead, we construct feedback with some summary information. As shown in Fig. 43, two types of information are included in the feedback: *intermediate causal graph* and *OLS regression result* of factors in the current estimation of a Markov blanket on the target variable. These two types of information can be drawn naturally from the COAT's intermediate results and are not complex, so LLMs can always access them and easily understand them. Intuitively, these factors would help LLMs better understand existing factors and propose further possible factors based on the provided dataset and its knowledge.

## K.2   Causal Graph and Discussion

**Result Interpretation**   In this case study, we are treating non-stationary climate data, while assuming the causal structure is invariant. Under this assumption, the non-stationarity is actually helpful for causal structure learning [51, 114]. To this end, we utilize the CD-NOD algorithm [51] to fully utilize the changing causal modules for better identifiability. CD-NOD would first identify the non-stationary nodes, whose conditional distribution given the causal parents are changing with time, and then use them for better causal structure recovery. Four factors are identified to be non-stationary, as shown in Fig. 44.

We have adjusted the nodes' names, shapes, and colors for better visualization. There are 14 nodes in total, with 13 factors identified by COAT. Each factor is a time series about a certain climate measurement above a specific level averaged over a specific region. For simplicity, we only considered instantaneous causal relations among those time series.

There are three regions identified to be relevant to the ENSO phenomenon:

- **Equatorial Pacific Region (Orange Nodes).** This region (5N-5S, 120W-280W) is one of the most active places about ENSO. It becomes significantly warm during the El Niño phase and becomes significantly cold during the La Niña phase [115].

- **Nino3 Region (Blue Nodes).** This region (5N-5S, 150W-90W) is one of the most classical regions to monitor the El Niño events by scientists. It is also used by humans to construct the target variables [110].

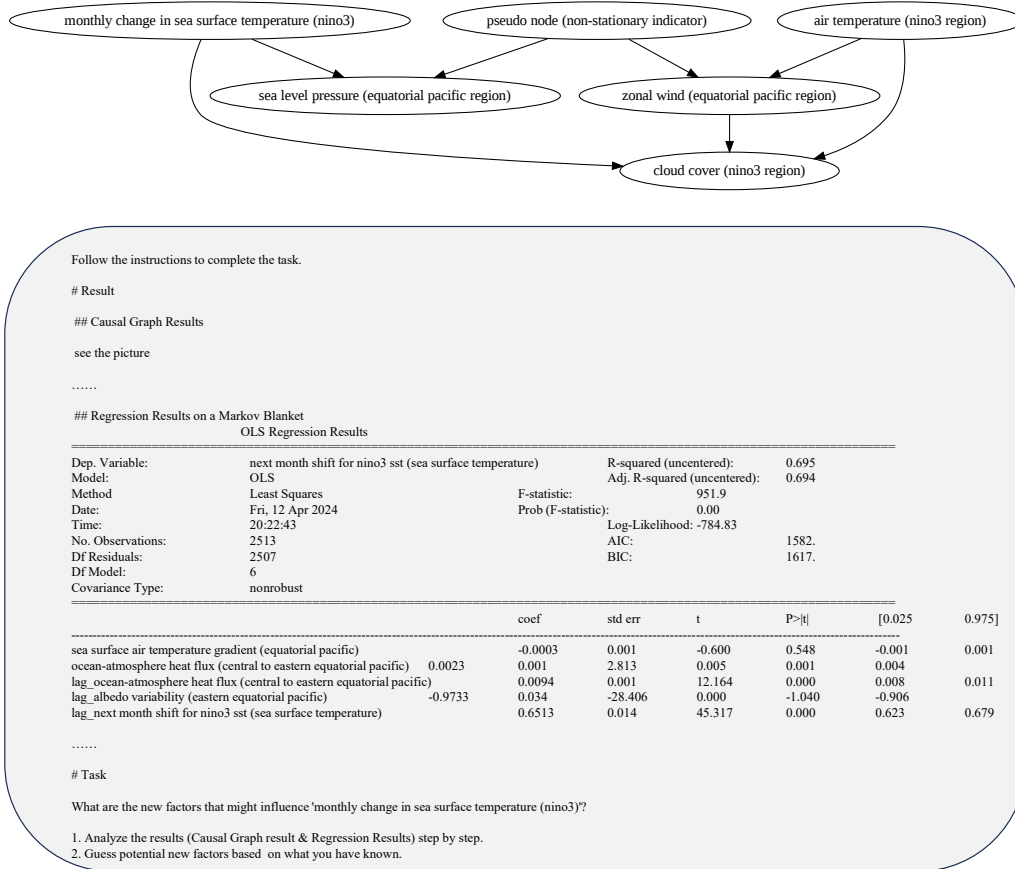

Figure 43: The prompt with feedback based on previous loop in the ENSO case study

- **South American Coastal Region (Green Nodes).** This region (0N-20S, 80W-60W) includes the Peruvian coastal up-welling system, and is noticed to be relevant to the ENSO cycle [116].

Some insights are delivered by paths in the output causal graph:

- **Sea level Pressure.** This factor is about the sea-level pressure on the equatorial Pacific region. The pressure gradient would influence the movement of warm water, and thus influence the sea surface temperature (SST) change [41]. In addition, pressure can influence the water evaporation and thus regulate through the water circulation. This also matches another indirect path Sea level Pressure → Sensible Heat Net Flux → Volumetric Soil Moisture → Cloud Cover → SST Change.

- **Momentum Flux, V-compoent.** This factor is about the vertical movement of air. It is crucial in driving atmospheric convection [41], and it is related to the Walker Circulation, which is an important component in the ENSO system[42]. Also, it could influence the change in sea level pressure and indirectly influence the SST change.

- **Cloud Cover.** The factor is the fraction of the sky covered by clouds in the NINO3 region. It could influence the SST Change through solar radiation as well as water circulation. It is confirmed to have a significant correlation [43] with ENSO events and plays an important role in the atmospheric circulation and hydrological cycle [44].

- **Soil Temperature.** This might be a novel hypothesis proposed by COAT, since we found no sufficient research to confirm this point to the best of our knowledge. This causal graph has also suggested two possible indirect mechanisms: (1) through *Volumetric Soil Moisture*

and *Cloud Cover*; and (2) through *Convective Precipitation Rate* and *Sea Level Pressure*. Therefore, this finding encourages more serious investigations of these hypotheses.

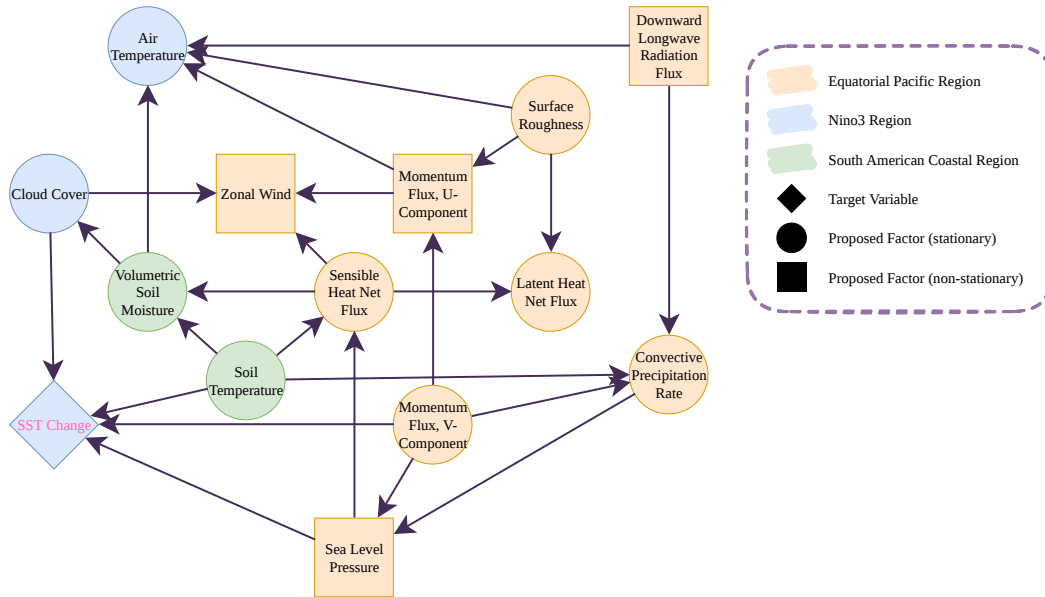

Figure 44: The final causal graph found by COAT in the ENSO case study

**Discussion on Additional assumption**  The constructed feedback contains the causal structure among existing factors and their OLS regression analysis on the target variable. By doing so, some additional assumptions are implicitly made. For example, it assumes a linear relation between factors and the target variables. More detailed theoretical analysis is out of the scope of this case study, and we left it for future work.

**Discussion on Hallucination**  Although we have clarified how to use tools in prompts, LLM can sometimes propose unsupported data requests. For example, only certain pressure levels are supported in the dataset. To this end, we include a Python code in the prompt, as listed in Lst. 3. This function can check whether the LLM's request is supported. We ask LLM to run this function in its code interpreter to make sure the proposed factors are valid.

```python
import pandas

def check_choice(measurement, level, detailed_level=None):

    meta = pd.read_csv('Measurement_Level_meta.csv')

    measurement_set = set(meta.Measurement.values)
    measurement_level_map = meta.groupby(['Measurement']).agg({"
    Level": lambda x: set(x)})
    detailed_level_range = {
        'Pressure Levels': [1000, 975, 950, 925, 900, 850, 800,
    750, 700, 650, 600, 550, 500, 450, 400, 350, 300, 250, 200, 150, 100,
     70, 50, 30, 20, 10, 5, 1],
        'Multiple Subsurface Levels':[0, 10, 40, 100],
        'Fixed Levels Above the Surface': [12, 20, 30, 50, 80,
    100, 150, 200, 250, 300, 500],
        'Isentropic Levels': [300, 330, 350]
    }

    if measurement not in measurement_set:
        return False, f"measurement error"
    if level not in measurement_level_map.loc[measurement,'Level'
    ]:
        return False, f"level error, only allowed: {
    measurement_level_map.loc[measurement,'Level']}"

    if level in detailed_level_range.keys():
        if detailed_level not in detailed_level_range[level]:
            return False, f"need detailed level in {
    detailed_level_range[level]}"

    return True, None
```

Listing 3: LLM is required to use this function to overcome hallucination

## L  Broader Impacts

This work focuses on fully leveraging the rich knowledge learned by LLMs during pre-training to facilitate causal discovery from unstructured data, with the hope of empowering broader applications and social benefits. Besides, this paper does not raise any ethical concerns. This study does not involve any human subjects, practices to data set releases, potentially harmful insights, methodologies and applications, potential conflicts of interest and sponsorship, discrimination/bias/fairness concerns, privacy and security issues, legal compliance, and research integrity issues.

